# Achieving Optimal Clustering in Gaussian Mixture Models with Anisotropic Covariance Structures

**Xin Chen**
Princeton University
xc5557@princeton.edu

**Anderson Ye Zhang**
University of Pennsylvania
ayz@wharton.upenn.edu

## Abstract

We study clustering under anisotropic Gaussian Mixture Models (GMMs), where covariance matrices from different clusters are unknown and are not necessarily the identity matrix. We analyze two anisotropic scenarios: homogeneous, with identical covariance matrices, and heterogeneous, with distinct matrices per cluster. For these models, we derive minimax lower bounds that illustrate the critical influence of covariance structures on clustering accuracy. To solve the clustering problem, we consider a variant of Lloyd's algorithm, adapted to estimate and utilize covariance information iteratively. We prove that the adjusted algorithm not only achieves the minimax optimality but also converges within a logarithmic number of iterations, thus bridging the gap between theoretical guarantees and practical efficiency.

## 1 Introduction

Clustering is a fundamentally important task in statistics and machine learning [7, 2]. The most widely recognized and extensively studied model for clustering is the Gaussian Mixture Model (GMM) [17, 19], which is formulated as

$$Y_j = \theta_{z_j^*}^* + \epsilon_j, \text{ where } \epsilon_j \overset{ind}{\sim} \mathcal{N}(0, \Sigma_{z_j^*}^*), \ \forall j \in [n].$$

Here $Y = (Y_1, \ldots, Y_n)$ are the observations with $n$ being the sample size. We define the set $[n] = \{1, 2, \ldots, n\}$. Assume $k$ is the known number of clusters. Let $\{\theta_a^*\}_{a \in [k]}$ represent the unknown centers, and $\Sigma_a^*$ denote the corresponding unknown covariance matrices. Define $z^* \in [k]^n$ as the cluster assignment vector, where for each index $j \in [n]$, the value of $z_j^*$ specifies which cluster the $j$-th data point is assigned to. The goal is to recover $z^*$ from $Y$. For any estimator $\hat{z}$, its clustering performance is measured by the misclustering error rate $h(\hat{z}, z^*)$, which will be introduced later in (4).

There has been increasing interest in theoretical and algorithmic analysis of clustering under GMMs. In a scenario where a GMM is isotropic, meaning that all covariance matrices $\{\Sigma_a^*\}_{a \in [k]}$ are equal to the identity matrix, [15] obtained the minimax rate for clustering, which takes the form of $\exp(-(1 + o(1))(\min_{a \neq b} \|\theta_a^* - \theta_b^*\|)^2/8)$, with respect to the misclustering error rate. A diverse range of methods has been explored in the context of the isotropic setting. Among these, Lloyd's algorithm [13] stands out as a particularly effective clustering algorithm, renowned for its extensive success in a myriad of disciplines. [15, 8] establish computational and statistical guarantees for the Lloyd's algorithm. Specifically, they showed it achieves the minimax optimal rates after a few iterations provided with some decent initialization. Another popular approach to clustering especially for high dimensional data is the spectral clustering [21, 18, 20], which is an umbrella term for clustering after a dimension reduction through a spectral decomposition. [14] proves the spectral clustering also achieves the optimality under the isotropic GMM. Semidefinite programming (SDP)

is also used for clustering by exploiting its low-rank structure, and its statistical properties have been studied in literature, for example, [5].

Despite the numerous compelling findings, most existing research primarily focuses on isotropic GMMs. The understanding of clustering in an anisotropic context, where the covariance matrices are not constrained to be identity matrices, remains relatively limited. Some studies, including [15, 5, 16, 1, 9, 24], present results for sub-Gaussian mixture models, wherein the errors $\epsilon_j$ are assumed to follow some sub-Gaussian distributions with the variance proxy $\sigma^2$. At first glance, it might appear that these results encompass the anisotropic case, as distributions of the form $\{\mathcal{N}(0, \Sigma_a^*)\}_{a \in [k]}$ are indeed sub-Gaussian distributions. However, from a minimax perspective, the least favorable scenario among all sub-Gaussian distributions with variance proxy $\sigma^2$—and thus the most challenging for clustering—is when the errors are distributed as $\mathcal{N}(0, \sigma^2 I)$. Therefore, the minimax rate for clustering under the sub-Gaussian mixture model essentially equals the one under the isotropic GMM, and methods like Lloyd's algorithm, which require no covariance matrix information, can be rate-optimal. As a result, the aforementioned findings primarily pertain to isotropic GMMs.

A few studies have explored the direction of clustering under anisotropic GMMs. [3] presents a polynomial-time clustering algorithm that provably performs well when Gaussian distributions are well-separated by hyperplanes. This idea is further developed in [11], which extends the approach to allow overlapping Gaussians, albeit only in two-cluster scenarios. [22] proposes a novel method for clustering under a balanced mixture of two elliptical distributions. They establish a provable upper bound on their clustering performance. Nevertheless, the fundamental limit of clustering under anisotropic GMMs, and whether a polynomial-time procedure can achieve it, remains unknown.

In this paper, we investigate the clustering task under two anisotropic GMMs. In Model 1, all covariance matrices are equal (i.e., homogeneous) to some unknown matrix $\Sigma^*$. Model 2 offers more flexibility, with covariance matrices that are unknown and not necessarily identical (i.e., heterogeneous). The contribution of this paper is two-fold, summarized as follows:

- Our first contribution is on the minimax rates. We obtain minimax lower bounds for clustering under anisotropic GMMs with respect to the misclustering error rate. We show they take the form of

$$\inf_{\hat{z}} \sup_{z^*} \mathbb{E} h(\hat{z}, z^*) \geq \exp\left(-(1 + o(1))\frac{(\text{signal-to-noise ratio})^2}{8}\right),$$

  where the signal-to-noise ratio under Model 1 is equal to $\min_{a,b \in [k]: a \neq b} \|(\theta_a^* - \theta_b^*)^T \Sigma^{* -\frac{1}{2}}\|$. The signal-to-noise ratio for Model 2 is more intricate and will be introduced in Section 3. For both models, we can see the minimax rates depend not only on the centers but also on the covariance matrices. This is different from the isotropic case, whose signal-to-noise ratio is $\min_{a \neq b} \|\theta_a^* - \theta_b^*\|$. Our results precisely capture the role that covariance matrices play in the clustering problem. This shows that covariance matrices impact the fundamental limits of the clustering problem through complex interactions with the centers, especially in Model 2. We obtain the minimax lower bounds by drawing connections with Linear Discriminant Analysis (LDA) [6] and Quadratic Discriminant Analysis (QDA).

- Our second and more important contribution is on the computational side. We give a computationally feasible procedure and rate-optimal algorithm for the anisotropic GMM. Lloyd's algorithm, developed for the isotropic case, is no longer optimal as it only considers distances among centers [3]. We study an adjusted Lloyd's algorithm which estimates the covariance matrices in each iteration and adjusts the clusters accordingly. It can also be seen as a hard EM algorithm [4]. Here, we modify the E-step of the soft EM by implementing a maximization step that directly assigns data points to clusters, rather than calculating probabilities. As an iterative algorithm, we demonstrate that it achieves the minimax lower bound within $\log n$ iterations. This offers both statistical and computational guarantees, serving as valuable guidance for practitioners. Specifically, if we let $z^{(t)}$ denote the output of the algorithm after $t$ iterations, it holds with high probability that

$$h(z^{(t)}, z^*) \leq \exp\left(-(1 + o(1))\frac{(\text{signal-to-noise ratio})^2}{8}\right),$$

  for all $t \geq \log n$. The algorithm can be initialized using popular methods like spectral clustering or Lloyd's algorithm. In our numerical studies, we demonstrate that our algorithm

significantly improves over the two aforementioned methods under anisotropic GMMs, and matches the optimal exponent specified in the minimax lower bound.

**Paper Organization.** The remaining paper is organized as follows. In Section 2, we study Model 1 where the covariance matrices are unknown but homogeneous. In Section 3, we consider Model 2 where covariance matrices are unknown and heterogeneous. For both cases, we establish the minimax lower bound for the clustering and give a computationally feasible and rate-optimal procedure. In Section 4, we provide a numerical comparison with other popular methods. Proofs are included in the supplement.

**Notation.** For any matrix $X \in \mathbb{R}^{d \times d}$, we denote $\lambda_1(X)$ as its smallest eigenvalue and $\lambda_d(X)$ as its largest eigenvalue. In addition, we denote $\|X\|$ as its operator norm. For any two vectors $u, v$ of the same dimension, we denote $\langle u, v \rangle = u^T v$ as their inner product. For any positive integer $d$, we denote $I_d$ as the $d \times d$ identity matrix. We denote $\mathcal{N}(\mu, \Sigma)$ as the normal distribution with mean $\mu$ and covariance matrix $\Sigma$. We denote $\mathbb{I}\{\cdot\}$ as the indicator function. For two positive sequences $\{a_n\}$ and $\{b_n\}$, $a_n \preceq b_n$ and $a_n = O(b_n)$ both mean $a_n \leq Cb_n$ for some constant $C > 0$ independent of $n$. We also write $a_n = o(b_n)$ or $\frac{b_n}{a_n} \to \infty$ when $\limsup_n \frac{a_n}{b_n} = 0$.

## 2 GMM with Unknown but Homogeneous Covariance Matrices

### 2.1 Model

We first consider the GMM where the covariance matrices of different clusters are unknown but are assumed to be equal to each other. Then the data-generating process can be displayed as follows:

**Model 1:** $$Y_j = \theta^*_{z^*_j} + \epsilon_j, \text{ where } \epsilon_j \overset{ind}{\sim} \mathcal{N}(0, \Sigma^*), \forall j \in [n]. \tag{1}$$

Throughout the paper, we call it *Model 1* for simplicity and to distinguish it from a different and more complicated one that will be introduced in Section 3. The goal is to recover the underlying cluster assignment vector $z^*$. If $\Sigma^*$ were known, then (1) can be converted into an isotropic GMM by a linear transformation $(\Sigma^*)^{-\frac{1}{2}} Y_j$. However, the unknown nature of $\Sigma^*$ makes clustering under this model more challenging than under isotropic GMMs.

**Signal-to-noise Ratio.** Define the signal-to-noise ratio

$$\text{SNR} = \min_{a,b \in [k]: a \neq b} \|(\theta^*_a - \theta^*_b)^T \Sigma^{* - \frac{1}{2}}\|, \tag{2}$$

which is a function of all the centers $\{\theta^*_a\}_{a \in [k]}$ and the covariance matrix $\Sigma^*$. As we will show later in Theorem 2.1, SNR captures the difficulty of the clustering problem and determines the minimax rate. We defer the geometric interpretation of SNR until after presenting Theorem 2.2.

A quantity closely related to SNR is the minimum distance among the centers. Define $\Delta$ as

$$\Delta = \min_{a,b \in [k]: a \neq b} \|\theta^*_a - \theta^*_b\|. \tag{3}$$

Then we can see SNR and $\Delta$ are of the same order if all eigenvalues of the covariance matrix $\Sigma^*$ are assumed to be constants. If $\Sigma^*$ is further assumed to be $\sigma^2 I_d$, then SNR equals $\Delta/\sigma$. As a result, in [15, 8, 14] where the isotropic GMMs are studied, $\Delta/\sigma$ plays the role of signal-to-noise ratio and appears in their rates. Since (2) represents a direct generalization, we refer to it as the signal-to-noise ratio for Model 1.

**Loss Function.** To measure the clustering performance, we consider the following loss function. For any $z, z^* \in [k]^n$, we define

$$h(z, z^*) = \min_{\psi \in \Psi} \frac{1}{n} \sum_{j=1}^{n} \mathbb{I}\{\psi(z_j) \neq z^*_j\}, \tag{4}$$

where $\Psi = \{\psi : \psi \text{ is a bijection from } [k] \text{ to } [k]\}$. Here, the minimum is taken over all permutations of $[k]$ to address the identifiability issues of the labels $1, 2, \ldots, k$. The loss function measures the

proportion of coordinates where $z$ and $z^*$ differ, modulo any permutation of label symbols. Thus, it is referred to as the misclustering error rate in this paper. Another loss that will be used is $\ell(z, z^*)$ defined as

$$\ell(z, z^*) = \sum_{j=1}^{n} \left\| \theta^*_{z_j} - \theta^*_{z^*_j} \right\|^2. \tag{5}$$

It measures the clustering performance of $z$ considering the distances among the true centers. It is related to $h(z, z^*)$ as $h(z, z^*) \leq \ell(z, z^*)/(n\Delta^2)$ and provides more information than $h(z, z^*)$. We will mainly use $\ell(z, z^*)$ in the technical analysis but will present results using $h(z, z^*)$ which is more interpretable.

## 2.2 Minimax Lower Bound

We first establish the minimax lower bound for the clustering problem under Model 1.

**Theorem 2.1.** *Under the assumption $\frac{SNR}{\sqrt{\log k}} \to \infty$, we have*

$$\inf_{\hat{z}} \sup_{z^* \in [k]^n} \mathbb{E}h(\hat{z}, z^*) \geq \exp\left( -(1 + o(1)) \frac{SNR^2}{8} \right). \tag{6}$$

*If $SNR = O(1)$ instead, we have $\inf_{\hat{z}} \sup_{z^* \in [k]^n} \mathbb{E}h(\hat{z}, z^*) \geq c$ for some constant $c > 0$.*

Theorem 2.1 allows the cluster numbers $k$ to grow with $n$ and shows that $SNR \to \infty$ is a necessary condition to have a consistent clustering. If $k$ is a constant, then $SNR \to \infty$ is also a sufficient condition. Theorem 2.1 holds for any arbitrary configurations of $\{\theta^*_a\}_{a \in [k]}$ and $\Sigma^*$, with the minimax lower bound depending on these through SNR. The parameter space is only for $z^*$ while $\{\theta^*_a\}_{a \in [k]}$ and $\Sigma^*$ are held fixed. Hence, (6) can be interpreted as a case-specific result, precisely capturing the explicit dependence of the minimax rates on $\{\theta^*_a\}_{a \in [k]}$ and $\Sigma^*$.

Theorem 2.1 is closely related to the LDA. If there are only two clusters with known centers and a covariance matrix, then estimating each $z^*_j$ becomes exactly the task of the LDA: we aim to determine from which of two normal distributions, each with a different mean but the same covariance matrix, the observation $Y_j$ is generated. In fact, this approach is also how Theorem 2.1 is proved: We first reduce the estimation problem of $z^*$ to two-point hypothesis testing for each individual $z^*_j$. The error of these tests is analyzed in Lemma A.1 using the LDA, and we then aggregate all these testing errors together.

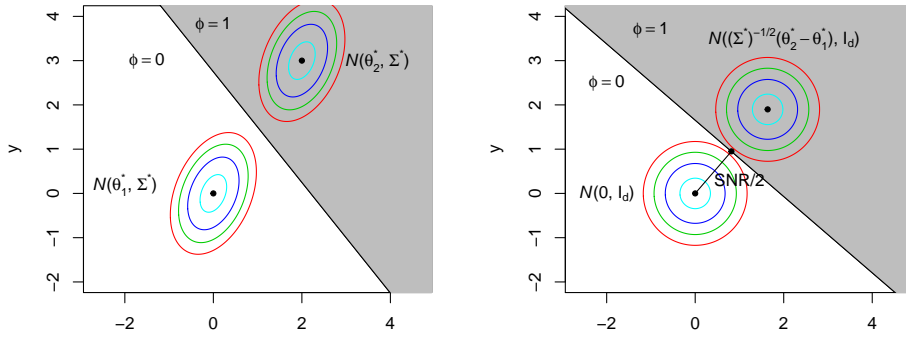

Figure 1: A geometric interpretation of SNR.

With the help of Lemma A.1, we have a geometric interpretation of SNR. In the left panel of Figure 1, we have two normal distributions $\mathcal{N}(\theta^*_1, \Sigma^*)$ and $\mathcal{N}(\theta^*_2, \Sigma^*)$ that $X$ follows. The black line represents the optimal testing procedure $\phi$ displayed in Lemma A.1, dividing the space into two half-spaces. To calculate the testing error, we can make the transformation $X' = (\Sigma^*)^{-\frac{1}{2}}(X - \theta^*_1)$ so that the two normal distributions become isotropic: $\mathcal{N}(0, I_d)$ and $\mathcal{N}((\Sigma^*)^{-\frac{1}{2}}(\theta^*_2 - \theta^*_1), I_d)$ as displayed in the right panel. Then the distance between the two centers is $\|(\Sigma^*)^{-\frac{1}{2}}(\theta^*_2 - \theta^*_1)\|$, and the distance from a center to the black curve is half of that. Then, the probability that $\mathcal{N}(0, I_d)$ falls within the

grayed area equals $\exp\left(-(1+o(1))\|(\Sigma^*)^{-\frac{1}{2}}(\theta_2^* - \theta_1^*)\|^2/8\right)$, according to Gaussian tail probability. As a result, $\|(\Sigma^*)^{-\frac{1}{2}}(\theta_2^* - \theta_1^*)\|$ is the effective distance between the two centers of $\mathcal{N}(\theta_1^*, \Sigma^*)$ and $\mathcal{N}(\theta_2^*, \Sigma^*)$ for the clustering problem, taking into account the geometry of the covariance matrix. Since we have multiple clusters, SNR defined in (2) can be interpreted as the minimum effective distance among the centers $\{\theta_a^*\}_{a \in [k]}$, considering the anisotropic structure of $\Sigma^*$. This measure captures the intrinsic difficulty of the clustering problem.

## 2.3 Rate-Optimal Adaptive Procedure

In this section, we give a computationally feasible and rate-optimal procedure for clustering under Model 1. Summarized in Algorithm 1, it is a variant of Lloyd's algorithm. Starting with an initial setup, it iteratively updates the estimates of the centers $\{\theta_a^*\}_{a \in [k]}$ (in (7)), the covariance matrix $\Sigma^*$ (in (8)), and the cluster assignment vector $z^*$ (in (9)). This algorithm differs from Lloyd's algorithm in that the latter is designed for isotropic GMMs and does not incorporate the covariance matrix update outlined in (8). Furthermore, (9) updates the estimation of $z_j^*$ using $\operatorname{argmin}_{a \in [k]}(Y_j - \theta_a^{(t)})^T(Y_j - \theta_a^{(t)})$ instead. To differentiate clearly, we refer to the classic form as the *vanilla Lloyd's algorithm* and our modified version, which accommodates the unknown and anisotropic covariance matrix, as the *adjusted Lloyd's algorithm*.

Algorithm 1 can also be interpreted as a hard EM algorithm. When applying Expectation Maximization (EM) to Model 1, the M step estimates the parameters $\{\theta_a^*\}_{a \in [k]}$ and $\Sigma^*$, while the E step estimates $z^*$. It turns out the updates on the parameters (7) - (8) are identical to those in the EM's M step. However, the update of $z^*$ in Algorithm 1 differs from that in the EM. Instead of computing a conditional expectation typical of the E step, the algorithm performs maximization in (9). As a result, Algorithm 1 effectively consists solely of M steps for both parameters and $z^*$, characterizing it as a hard EM algorithm.

---

**Algorithm 1:** Adjusted Lloyd's Algorithm for Model 1.

---

**Input:** Data $Y$, number of clusters $k$, an initialization $z^{(0)}$, number of iterations $T$.
**Output:** $z^{(T)}$

1 **for** $t = 1, \ldots, T$ **do**

2 $\quad$ Update the centers:

$$\theta_a^{(t)} = \frac{\sum_{j \in [n]} Y_j \mathbb{I}\left\{z_j^{(t-1)} = a\right\}}{\sum_{j \in [n]} \mathbb{I}\left\{z_j^{(t-1)} = a\right\}}, \quad \forall a \in [k]. \tag{7}$$

3 $\quad$ Update the covariance matrix:

$$\Sigma^{(t)} = \frac{\sum_{a \in [k]} \sum_{j \in [n]}(Y_j - \theta_a^{(t)})(Y_j - \theta_a^{(t)})^T \mathbb{I}\left\{z_j^{(t-1)} = a\right\}}{n}. \tag{8}$$

4 $\quad$ Update the cluster assignment vector:

$$z_j^{(t)} = \operatorname*{argmin}_{a \in [k]}(Y_j - \theta_a^{(t)})^T(\Sigma^{(t)})^{-1}(Y_j - \theta_a^{(t)}), \quad \forall j \in [n]. \tag{9}$$

---

In Theorem 2.2, we give a computational and statistical guarantee of Algorithm 1. We show that starting from a decent initialization, within $\log n$ iterations, Algorithm 1 achieves the error rate $\exp\left(-(1+o(1))\mathrm{SNR}^2/8\right)$ which matches the minimax lower bound given in Theorem 2.1. As a result, Algorithm 1 is a rate-optimal procedure. In addition, the algorithm is fully adaptive to the unknown $\{\theta_a^*\}_{a \in [k]}$ and $\Sigma^*$. The sole piece of information presumed to be known is $k$, the number of clusters, as commonly assumed in clustering literature [15, 8, 14]. The theorem also shows that the number of iterations needed to achieve the optimal rate is at most $\log n$, providing implementation guidance to practitioners.

**Theorem 2.2.** *Assume $k = O(1)$, $d = O(\sqrt{n})$, and $\min_{a \in [k]} \sum_{j=1}^n \mathbb{I}\{z_j^* = a\} \geq \frac{\alpha n}{k}$ for some constant $\alpha > 0$. Assume SNR $\to \infty$ and $\lambda_d(\Sigma^*)/\lambda_1(\Sigma^*) = O(1)$. For Algorithm 1, suppose $z^{(0)}$ satisfies $\ell(z^{(0)}, z^*) = o(n)$ with probability at least $1 - \eta$. Then with probability at least $1 - \eta - n^{-1} - \exp(-SNR)$, we have*

$$h(z^{(t)}, z^*) \leq \exp\left(-(1 + o(1))\frac{SNR^2}{8}\right), \quad \text{for all } t \geq \log n.$$

We make the following remarks on the assumptions of Theorem 2.2: When $k$ is constant, the assumption that SNR $\to \infty$ is a necessary condition for consistent recovery of $z^*$, as outlined in the minimax lower bound presented in Theorem 2.1. The assumption on $\Sigma^*$ ensures that the covariance matrix is well-conditioned. The dimensionality $d$ is assumed to be $O(\sqrt{n})$, a stronger assumption than in [15, 8, 14], where $d = O(n)$ is sufficient. This is because, unlike these studies, our work requires estimating the covariance matrix $\Sigma^*$ and controlling the estimation error $\|\Sigma^{(t)} - \Sigma^*\|$.

Theorem 2.2 needs a decent initialization $z^{(0)}$ in the sense that it is sufficiently close to the ground truth such that $\ell(z^{(0)}, z^*) = o(n)$. This is because our theoretical analysis requires the initialization being within a specific proximity to the true parameters. The requirement can be fulfilled by simple procedures. An example is the vanilla Lloyd's algorithm whose performance is studied in [15, 8]. Though [15, 8] are for isotropic GMMs, their results can be extended to sub-Gaussian mixture models with nearly identical proof. Since $\epsilon_j$ are sub-Gaussian random variables with proxy variance $\lambda_d(\Sigma^*)$, [8] implies the vanilla Lloyd's algorithm output $\hat{z}$ satisfies $\ell(\hat{z}, z^*) \leq n \exp(-(1 + o(1))\Delta^2/(8\lambda_d(\Sigma^*)))$ with probability at least $1 - \exp(-\Delta/\sqrt{\lambda_d(\Sigma^*)}) - n^{-1}$, under the assumption that $\Delta^2/(k^2(kd/n + 1)\lambda_d(\Sigma^*)) \to \infty$. Then we have $\ell(\hat{z}, z^*) = o(n)$ with high probability under the assumptions of Theorem 2.2, and hence it can be used as an initialization for the algorithm.

# 3 GMM with Unknown and Heterogeneous Covariance Matrices

## 3.1 Model

In this section, we study the GMM where the covariance matrices of each cluster are unknown and not necessarily equal to each other. The data-generation process can be displayed as follows,

**Model 2:** $$Y_j = \theta_{z_j^*}^* + \epsilon_j, \text{ where } \epsilon_j \overset{ind}{\sim} \mathcal{N}(0, \Sigma_{z_j^*}^*), \forall j \in [n]. \tag{10}$$

We refer to this as *Model 2* throughout the paper to distinguish it from Model 1, as discussed in Section 2. The key difference between (10) and (1) is that here we have distinct covariance matrices $\{\Sigma_a^*\}_{a \in [k]}$ for each cluster, instead of a single shared $\Sigma^*$. We use the same loss function as defined in (4).

**Signal-to-noise Ratio.** The signal-to-noise ratio for Model 2 is defined as follows. We use the notation SNR$'$ to distinguish it from the SNR used for Model 1. Compared to SNR, SNR$'$ is much more complicated and does not have an explicit formula. We first define a set $B_{a,b} \subset \mathbb{R}^d$ for any $a, b \in [k]$ such that $a \neq b$:

$$B_{a,b} = \left\{ x \in \mathbb{R}^d : x^T \Sigma_a^{*\frac{1}{2}} \Sigma_b^{*-1}(\theta_a^* - \theta_b^*) + \frac{1}{2} x^T \left( \Sigma_a^{*\frac{1}{2}} \Sigma_b^{*-1} \Sigma_a^{*\frac{1}{2}} - I_d \right) x \right.$$

$$\left. \leq -\frac{1}{2}(\theta_a^* - \theta_b^*)^T \Sigma_b^{*-1}(\theta_a^* - \theta_b^*) + \frac{1}{2} \log |\Sigma_a^*| - \frac{1}{2} \log |\Sigma_b^*| \right\}.$$

We then define SNR$'_{a,b} = 2 \min_{x \in B_{a,b}} \|x\|$ and

$$\text{SNR}' = \min_{a,b \in [k]: a \neq b} \text{SNR}'_{a,b}. \tag{11}$$

The form of SNR$'$ is closely connected to the testing error of the QDA, which we will give in Lemma 3.1. The interpretation of the SNR$'$, particularly from a geometric perspective, will be

deferred until after the presentation of Lemma 3.1. Here let us consider a few special cases where we are able to simplify SNR$'$: (1) When $\Sigma_a^* = \Sigma^*$ for all $a \in [k]$, by simple algebra, we have SNR$'_{a,b} = \|(\theta_a^* - \theta_b^*)^T \Sigma^{*-\frac{1}{2}}\|$ for any $a, b \in [k]$ such that $a \neq b$. Hence, SNR$'$ = SNR and Model 2 effectively reduces to Model 1. (2) When $\Sigma_a^* = \sigma_a^2 I_d$ for any $a \in [k]$ where $\sigma_1, \ldots, \sigma_k > 0$ are large constants, we have SNR$'_{a,b}$, SNR$'_{b,a}$ both close to $2\|\theta_a^* - \theta_b^*\|/(\sigma_a + \sigma_b)$. From these examples, we can see SNR$'$ is determined by both the centers $\{\theta_a^*\}_{a \in [k]}$ and the covariance matrices $\{\Sigma_a^*\}_{a \in [k]}$.

## 3.2 Minimax Lower Bound

We first establish the minimax lower bound for the clustering problem under Model 2.

**Theorem 3.1.** *Assume $d = O(1)$ and $\max_{a,b \in [k]} \lambda_d(\Sigma_a^*)/\lambda_1(\Sigma_b^*) = O(1)$. Under the assumption $\frac{SNR'}{\sqrt{\log k}} \to \infty$, we have*

$$\inf_{\hat{z}} \sup_{z^* \in [k]^n} \mathbb{E}h(\hat{z}, z^*) \geq \exp\left(-(1+o(1))\frac{SNR'^2}{8}\right).$$

*If SNR$'$ = $O(1)$ instead, we have $\inf_{\hat{z}} \sup_{z^* \in [k]^n} \mathbb{E}h(\hat{z}, z^*) \geq c$ for some constant $c > 0$.*

Although the statement of Theorem 3.1 appears similar to that of Theorem 2.1, the two minimax lower bounds differ due to the varying dependencies of the centers and covariance matrices on SNR$'$ versus SNR. Using the same argument as in Section 2.2, the minimax lower bound established in Theorem 3.1 closely relates to the QDA between two normal distributions with different means and different covariance matrices.

**Lemma 3.1** (Testing Error for the QDA). *Consider two hypotheses $\mathbb{H}_0 : X \sim \mathcal{N}(\theta_1^*, \Sigma_1^*)$ and $\mathbb{H}_1 : X \sim \mathcal{N}(\theta_2^*, \Sigma_2^*)$. Define a testing procedure*

$$\phi = \mathbb{I}\left\{\log|\Sigma_1^*| + (x - \theta_1^*)^T (\Sigma_1^*)^{-1}(x - \theta_1^*) \geq \log|\Sigma_2^*| + (x - \theta_2^*)^T (\Sigma_2^*)^{-1}(x - \theta_2^*)\right\}.$$

*Then we have $\inf_{\hat{\phi}}(\mathbb{P}_{\mathbb{H}_0}(\hat{\phi} = 1) + \mathbb{P}_{\mathbb{H}_1}(\hat{\phi} = 0)) = \mathbb{P}_{\mathbb{H}_0}(\phi = 1) + \mathbb{P}_{\mathbb{H}_1}(\phi = 0)$. Assume $d = O(1)$ and $\max_{a,b \in \{1,2\}} \lambda_d(\Sigma_a^*)/\lambda_1(\Sigma_b^*) = O(1)$. If $\min\{SNR'_{1,2}, SNR'_{2,1}\} \to \infty$, we have*

$$\inf_{\hat{\phi}}(\mathbb{P}_{\mathbb{H}_0}(\hat{\phi} = 1) + \mathbb{P}_{\mathbb{H}_1}(\hat{\phi} = 0)) \geq \exp\left(-(1+o(1))\frac{\min\{SNR'_{1,2}, SNR'_{2,1}\}^2}{8}\right).$$

*Otherwise, $\inf_{\hat{\phi}}(\mathbb{P}_{\mathbb{H}_0}(\hat{\phi} = 1) + \mathbb{P}_{\mathbb{H}_1}(\hat{\phi} = 0)) \geq c$ for some constant $c > 0$.*

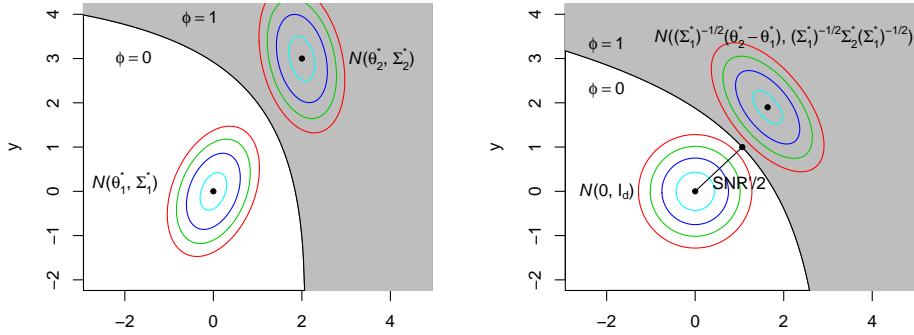

Figure 2: A geometric interpretation of SNR$'$.

Lemma 3.1 provides a geometric interpretation of SNR$'$. In the left panel of Figure 2, we have two normal distributions $\mathcal{N}(\theta_1^*, \Sigma_1^*)$ and $\mathcal{N}(\theta_2^*, \Sigma_2^*)$ from which $X$ can be generated, and the black curve represents the optimal testing procedure $\phi$, as detailed in Lemma 3.1. Since $\Sigma_1^*$ is not necessarily equal to $\Sigma_2^*$, the black curve is not necessarily a straight line. If $\mathbb{H}_0$ is true, the probability that $X$ is incorrectly classified occurs when $X$ falls into the gray area, represented by $\mathbb{P}_{\mathbb{H}_0}(\phi = 1)$. To

calculate this, we transform $X$ to $X' = (\Sigma_1^*)^{-\frac{1}{2}}(X - \theta_1^*)$, standardizing the first distribution. Then, as displayed in the right panel of Figure 2, the two distributions become $\mathcal{N}(0, I_d)$ and $\mathcal{N}((\Sigma_1^*)^{-\frac{1}{2}}(\theta_2^* - \theta_1^*), (\Sigma_1^*)^{-\frac{1}{2}}\Sigma_2^*(\Sigma_1^*)^{-\frac{1}{2}})$, and the optimal testing procedure $\phi$ becomes $\mathbb{I}\{X' \in B_{1,2}\}$. As a result, in the right panel of Figure 2, $B_{1,2}$ represents the space colored by gray, and the black curve is its boundary. Then $\mathbb{P}_{\mathbb{H}_0}(\phi = 1)$ is equal to $\mathbb{P}(\mathcal{N}(0, I_d) \in B_{1,2})$. Under the assumption $d = O(1)$ and $\max_{a,b \in \{1,2\}} \lambda_d(\Sigma_a^*)/\lambda_1(\Sigma_b^*) = O(1)$, in Lemma C.10, we can show $\mathbb{P}(\mathcal{N}(0, I_d) \in B_{1,2}) = \exp(-(1 + o(1))\mathrm{SNR}_{1,2}'^2/8)$. As a result, $\mathrm{SNR}'$ can be interpreted as the minimum effective distance among the centers $\{\theta_a^*\}_{a \in [k]}$, considering the anisotropic and heterogeneous structure of $\{\Sigma_a^*\}_{a \in [k]}$, and it captures the intrinsic difficulty of the clustering problem under Model 2.

### 3.3 Optimal Adaptive Procedure

In this section, we give a computationally feasible and rate-optimal procedure for clustering under Model 2. Similar to Algorithm 1, Algorithm 2 is a variant of Lloyd's algorithm, adjusted to accommodate unknown and heterogeneous covariance matrices. It can also be interpreted as a hard EM algorithm under Model 2. Algorithm 2 differs from Algorithm 1 in (13) and (14), as now there are $k$ covariance matrices instead of a common one.

---

**Algorithm 2:** Adjusted Lloyd's Algorithm for Model 2.

**Input:** Data $Y$, number of clusters $k$, an initialization $z^{(0)}$, number of iterations $T$.
**Output:** $z^{(T)}$

1 **for** $t = 1, \ldots, T$ **do**

2      Update the centers:

$$\theta_a^{(t)} = \frac{\sum_{j \in [n]} Y_j \mathbb{I}\left\{z_j^{(t-1)} = a\right\}}{\sum_{j \in [n]} \mathbb{I}\left\{z_j^{(t-1)} = a\right\}}, \quad \forall a \in [k]. \tag{12}$$

3      Update the covariance matrices:

$$\Sigma_a^{(t)} = \frac{\sum_{j \in [n]} (Y_j - \theta_a^{(t)})(Y_j - \theta_a^{(t)})^T \mathbb{I}\left\{z_j^{(t-1)} = a\right\}}{\sum_{j \in [n]} \mathbb{I}\left\{z_j^{(t-1)} = a\right\}}, \quad \forall a \in [k]. \tag{13}$$

4      Update the cluster assignment vector:

$$z_j^{(t)} = \underset{a \in [k]}{\arg\min}(Y_j - \theta_a^{(t)})^T (\Sigma_a^{(t)})^{-1}(Y_j - \theta_a^{(t)}) + \log|\Sigma_a^{(t)}|, \quad \forall j \in [n]. \tag{14}$$

---

In Theorem 3.2, we give a computational and statistical guarantee for Algorithm 2. We demonstrate that, with proper initialization, Algorithm 2 achieves the minimax lower bound within $\log n$ iterations. The assumptions needed in Theorem 3.2 are similar to those in Theorem 2.2, except that we require stronger assumptions on the dimensionality $d$ since now we have $k$ (instead of one) covariance matrices to be estimated. In addition, by assuming $\max_{a,b \in [k]} \lambda_d(\Sigma_a^*)/\lambda_1(\Sigma_b^*) = O(1)$, we ensure not only that each of the $k$ covariance matrices is well-conditioned but also that they are comparable to one another.

**Theorem 3.2.** *Assume $k = O(1)$, $d = O(1)$, and $\min_{a \in [k]} \sum_{j=1}^n \mathbb{I}\{z_j^* = a\} \geq \frac{\alpha n}{k}$ for some constant $\alpha > 0$. Assume $\mathrm{SNR}' \to \infty$ and $\max_{a,b \in [k]} \lambda_d(\Sigma_a^*)/\lambda_1(\Sigma_b^*) = O(1)$. For Algorithm 2, suppose $z^{(0)}$ satisfies $\ell(z^{(0)}, z^*) = o(n)$ with probability at least $1 - \eta$. Then with probability at least $1 - \eta - 5n^{-1} - \exp(-\mathrm{SNR}')$, we have*

$$h(z^{(t)}, z^*) \leq \exp\left(-(1 + o(1))\frac{\mathrm{SNR}'^2}{8}\right), \quad \text{for all } t \geq \log n.$$

The vanilla Lloyd's algorithm can be used as the initialization for Algorithm 2. This is because Model 2 is also a sub-Gaussian mixture model. By the same argument as in Section 2.3, the output of the

vanilla Lloyd's algorithm $\hat{z}$ satisfies $\ell(\hat{z}, z^*) = o(n)$ with high probability under the assumptions of Theorem 3.2.

We conclude this section with a time complexity analysis of Algorithm 2. Compared to the vanilla Lloyd's algorithm, our method introduces additional computational overhead due to the need for computing the inverse and determinant of covariance matrices. Specifically, the time complexity of Algorithm 2 is $O(nkd^3T)$. In contrast, the vanilla Lloyd's algorithm has a lower time complexity of $O(nkdT)$. The increase in complexity stems from matrix operations in $d$ dimensions, as both matrix inversion and determinant computation scale as $O(d^3)$.

## 4  Numerical Studies

In this section, we compare the performance of our methods with other popular clustering methods on synthetic and real datasets under different settings.

**Model 1.**   The first simulation is designed for the GMM with unknown but homogeneous covariance matrices (i.e., Model 1). We independently generate $n = 1200$ samples with dimension $d = 50$ from $k = 30$ clusters. Each cluster has 40 samples. We set $\Sigma^* = U^T \Lambda U$, where $\Lambda$ is a $50 \times 50$ diagonal matrix with diagonal elements selected from 0.5 to 8 with equal space and $U$ is a randomly generated orthogonal matrix. The centers $\{\theta_a^*\}_{a \in [n]}$ are orthogonal to each other with $\|\theta_1^*\| = \ldots = \|\theta_{30}^*\| = 9$. We consider four popular clustering methods: (1) the spectral clustering method in [14] (denoted as "spectral"), (2) the vanilla Lloyd's algorithm in [15] (denoted as "vanilla Lloyd"), (3) Algorithm 1 initialized by the spectral clustering (denoted as "spectral + Alg 1"), and (4) Algorithm 1 initialized by the vanilla Lloyd (denoted as "vanilla Lloyd + Alg 1"). The comparison is presented in the left panel of Figure 3.

**Model 2.**   We also compare the performances of four methods (spectral, vanilla Lloyd, spectral + Alg 2, and vanilla Lloyd + Alg 2) for the GMM with unknown and heterogeneous covariance matrices (i.e., Model 2). In this case, we take $n = 1200$, $k = 2$, and $d = 9$. We set $\Sigma_1^* = I_d$ and $\Sigma_2^* = \Lambda_2$, a diagonal matrix where the first diagonal entry is 0.5 and the remaining entries are 5. We set the cluster sizes to be 900 and 300, respectively. To simplify the calculation of SNR$'$, we set $\theta_1^* = 0$ and $\theta_2^* = 5e_1$, with $e_1$ being the vector that has a 1 in its first entry and 0s elsewhere. The comparison is presented in the right panel of Figure 3.

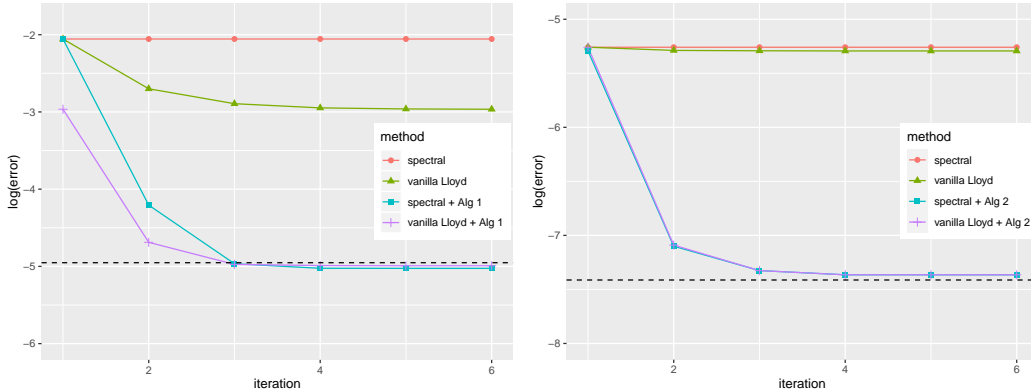

Figure 3: Left: Performance of Algorithm 1 compared with other methods under Model 1. Right: Performance of Algorithm 2 compared with other methods under Model 2.

In Figure 3, the $x$-axis is the number of iterations and the $y$-axis is the logarithm of the misclustering error rate, i.e., $\log(h)$. Each of the curves plotted is an average of 100 independent trials. We can see both Algorithm 1 and Algorithm 2 outperform the spectral clustering and the vanilla Lloyd's algorithm significantly. Additionally, the dashed lines in the left and right panels represent the optimal exponents $-\text{SNR}^2/8$ and $-\text{SNR}'^2/8$ of the minimax bounds, respectively. It is observed that both Algorithm 1 and Algorithm 2 meet these benchmarks after three iterations. This justifies the conclusion that both algorithms are rate-optimal.

**Real Data.** To further demonstrate the effectiveness of our methods, we conduct experiments using the Fashion-MNIST dataset [23]. In the first analysis, we use a total of 12,000 28×28 grayscale images, consisting of 6,000 images each from the T-shirt/top class and the Trouser class. The left panel of Figure 4 gives a visualization of the data points using their first two principal components, showing the anisotropic and heterogeneous covariance structures. Since a large number of pixels have zero across most images, we apply PCA to reduce dimensionality from 784 to 50 by retaining the top 50 principal components. Our Algorithm 2 achieves a misclustering error of 5.71%, outperforming the vanilla Lloyd's algorithm, which has an error of 8.24%. In the second analysis, we incorporate an additional class, the Ankle boot class, increasing the total to 18,000 images across three classes. Following the same preprocessing steps, the visualization of the dataset's structure in the right panel of Figure 4 again confirms the presence of anisotropic and heterogeneous covariances. Here, Algorithm 2 achieves an error of 3.97%, an improvement over the 5.64% error rate observed with the vanilla Lloyd's algorithm.

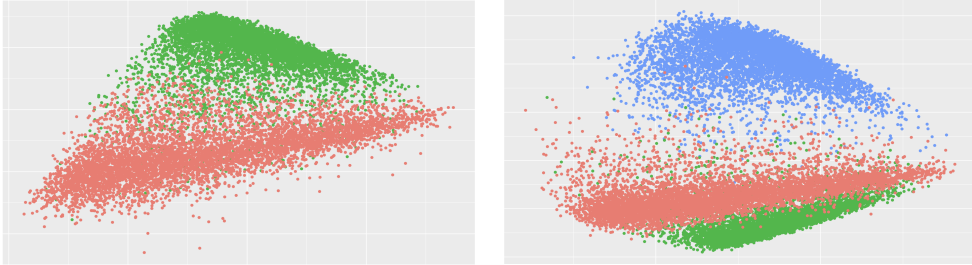

Figure 4: Visualization of the Fashion-MNIST dataset using the first two principal components. The data points are color-coded to indicate class membership: Red represents the T-shirt/top class, green denotes the Trouser class, and blue signifies the Ankle boot class. This illustration shows the existence of anisotropic and heterogeneous covariance structures.

## 5   Conclusion

This paper focuses on clustering methods and theory for GMMs, with anisotropic covariance structures, presenting new minimax bounds and an adjusted Lloyd's algorithm tailored for varying covariance structures. Our theoretical and empirical analyses demonstrate the algorithm's ability to achieve optimality within a logarithmic number of iterations. Despite these advances, our results have some limitations that are worth addressing in future work:

1. **High-Dimensional Settings:** Current results are restricted to dimensions $d$ growing at a rate slower than $n$, specifically $d = O(\sqrt{n})$ as stated in Theorem 2.2. Section 3 further requires a stronger assumption $d = O(1)$. These constraints stem from technical challenges in estimating covariance matrices accurately and in controlling matrix determinant. Adopting more sophisticated analytical tools could potentially relax these bounds to $d = O(n)$. In scenarios where $d$ exceeds $n$, the misclustering error deviates from the simpler exponential decay observed under isotropic GMMs, as shown in [16]. This suggests that our model might also exhibit similar complexities, warranting further exploration into the technique used in [16] for potential extensions.

2. **Ill-Conditioned Covariance Structures:** Our analysis relies on the assumption of well-conditioned covariance matrices, where $\max_{a,b \in [k]} \lambda_d(\Sigma_a^*)/\lambda_1(\Sigma_b^*) = O(1)$. This condition is crucial for the current analytical framework, as it helps manage the estimation errors of covariance matrices and their inverses. While more advanced techniques may allow for a relaxation of this assumption, handling ill-conditioned or degenerate covariance matrices remains challenging, particularly due to the difficulty of working with matrix inverses in such cases. While minimax lower bounds suggest that clustering is still possible even when the covariance matrix is degenerate, it raises computational challenges for our current algorithms. This highlights the need for developing new algorithms that can function effectively under less restrictive conditions.

# References

[1] Emmanuel Abbe, Jianqing Fan, and Kaizheng Wang. An $\ell_p$ theory of PCA and spectral clustering. *The Annals of Statistics*, 50(4):2359–2385, 2022.

[2] Christopher M Bishop. *Pattern recognition and machine learning*. springer, 2006.

[3] S Charles Brubaker and Santosh S Vempala. Isotropic PCA and affine-invariant clustering. In *Building Bridges*, pages 241–281. Springer, 2008.

[4] Arthur P Dempster, Nan M Laird, and Donald B Rubin. Maximum likelihood from incomplete data via the EM algorithm. *Journal of the Royal Statistical Society: Series B (Methodological)*, 39(1):1–22, 1977.

[5] Yingjie Fei and Yudong Chen. Hidden integrality of SDP relaxations for sub-Gaussian mixture models. In *Conference On Learning Theory*, pages 1931–1965. PMLR, 2018.

[6] Ronald A Fisher. The use of multiple measurements in taxonomic problems. *Annals of eugenics*, 7(2):179–188, 1936.

[7] Jerome Friedman, Trevor Hastie, and Robert Tibshirani. *The elements of statistical learning*, volume 1. Springer series in statistics New York, 2001.

[8] Chao Gao and Anderson Y Zhang. Iterative algorithm for discrete structure recovery. *The Annals of Statistics*, 50(2):1066–1094, 2022.

[9] Christophe Giraud and Nicolas Verzelen. Partial recovery bounds for clustering with the relaxed $k$-means. *Mathematical Statistics and Learning*, 1(3):317–374, 2019.

[10] Daniel Hsu, Sham Kakade, Tong Zhang, et al. A tail inequality for quadratic forms of subgaussian random vectors. *Electronic Communications in Probability*, 17, 2012.

[11] Adam Tauman Kalai, Ankur Moitra, and Gregory Valiant. Efficiently learning mixtures of two Gaussians. In *Proceedings of the forty-second ACM symposium on Theory of computing*, pages 553–562, 2010.

[12] Beatrice Laurent and Pascal Massart. Adaptive estimation of a quadratic functional by model selection. *The Annals of Statistics*, pages 1302–1338, 2000.

[13] Stuart Lloyd. Least squares quantization in PCM. *IEEE transactions on information theory*, 28(2):129–137, 1982.

[14] Matthias Löffler, Anderson Y Zhang, and Harrison H Zhou. Optimality of spectral clustering in the Gaussian mixture model. *The Annals of Statistics*, 49(5):2506–2530, 2021.

[15] Yu Lu and Harrison H Zhou. Statistical and computational guarantees of Lloyd's algorithm and its variants. *arXiv preprint arXiv:1612.02099*, 2016.

[16] Mohamed Ndaoud. Sharp optimal recovery in the two component Gaussian mixture model. *The Annals of Statistics*, 50(4):2096–2126, 2022.

[17] Karl Pearson. Contributions to the mathematical theory of evolution. *Philosophical Transactions of the Royal Society of London. A*, 185:71–110, 1894.

[18] Daniel A Spielman and Shang-Hua Teng. Spectral partitioning works: Planar graphs and finite element meshes. In *Proceedings of 37th Conference on Foundations of Computer Science*, pages 96–105. IEEE, 1996.

[19] D Michael Titterington, Adrian FM Smith, and Udi E Makov. *Statistical analysis of finite mixture distributions*. Wiley,, 1985.

[20] S. Vempala and G. Wang. A spectral algorithm for learning mixture models. *J. Comput. Syst. Sci.*, 68(4):841–860, 2004.

[21] Ulrike Von Luxburg. A tutorial on spectral clustering. *Statistics and computing*, 17(4):395–416, 2007.

[22] Kaizheng Wang, Yuling Yan, and Mateo Díaz. Efficient clustering for stretched mixtures: Landscape and optimality. *Advances in Neural Information Processing Systems*, 33:21309–21320, 2020.

[23] Han Xiao, Kashif Rasul, and Roland Vollgraf. Fashion-mnist: a novel image dataset for benchmarking machine learning algorithms, 2017.

[24] Anderson Y Zhang and Harrison H Zhou. Leave-one-out singular subspace perturbation analysis for spectral clustering. *arXiv preprint arXiv:2205.14855*, 2022.

The appendices are organized as follows. Appendix A is dedicated to proving the results in Section 2. To be more specific, we prove the lower bound, Theorem 2.1, in Appendix A.1 and the upper bound, Theorem 2.2, in Appendix A.2. For the upper bound proof, we first give a high-level idea in Appendix A.2.1, followed by a detailed proof in Appendix A.2.2. Appendix B includes proofs of the results in Section 3: the proof of the lower bound, Theorem 3.1, is in Appendix B.1, and the proof of the upper bound, Theorem 3.2, is in Appendix B.2. We include all technical lemmas and their proofs in Appendix C.

# A  Proofs in Section 2

## A.1  Proofs for the Lower Bound

In the following lemma, we give a sharp and explicit formula for the testing error of the LDA. Here we have two normal distributions $\mathcal{N}(\theta_1^*, \Sigma^*)$ and $\mathcal{N}(\theta_2^*, \Sigma^*)$ and an observation $X$ that is generated from one of them. We are interested in estimating from which distribution the observation is drawn. By the Neyman-Pearson lemma, it is known that the likelihood ratio test $\mathbb{I}\left\{2(\theta_2^* - \theta_1^*)^T(\Sigma^*)^{-1}X \geq \theta_2^{*T}(\Sigma^*)^{-1}\theta_2^* - \theta_1^{*T}(\Sigma^*)^{-1}\theta_1^*\right\}$ is the optimal testing procedure. Then by using the Gaussian tail probability, we are able to obtain the optimal testing error, with its lower bound given in Lemma A.1.

**Lemma A.1** (Testing Error for the LDA). *Consider two hypotheses $\mathbb{H}_0 : X \sim \mathcal{N}(\theta_1^*, \Sigma^*)$ and $\mathbb{H}_1 : X \sim \mathcal{N}(\theta_2^*, \Sigma^*)$. Define a testing procedure*

$$\phi = \mathbb{I}\left\{2(\theta_2^* - \theta_1^*)^T(\Sigma^*)^{-1}X \geq \theta_2^{*T}(\Sigma^*)^{-1}\theta_2^* - \theta_1^{*T}(\Sigma^*)^{-1}\theta_1^*\right\}.$$

*Then $\inf_{\hat{\phi}}(\mathbb{P}_{\mathbb{H}_0}(\hat{\phi} = 1) + \mathbb{P}_{\mathbb{H}_1}(\hat{\phi} = 0)) = \mathbb{P}_{\mathbb{H}_0}(\phi = 1) + \mathbb{P}_{\mathbb{H}_1}(\phi = 0)$. If $\|(\theta_2^* - \theta_1^*)^T(\Sigma^*)^{-\frac{1}{2}}\| \to \infty$, we have*

$$\inf_{\hat{\phi}}(\mathbb{P}_{\mathbb{H}_0}(\hat{\phi} = 1) + \mathbb{P}_{\mathbb{H}_1}(\hat{\phi} = 0)) \geq \exp\left(-(1 + o(1))\frac{\|(\theta_2^* - \theta_1^*)^T(\Sigma^*)^{-\frac{1}{2}}\|^2}{8}\right).$$

*Otherwise, $\inf_{\hat{\phi}}(\mathbb{P}_{\mathbb{H}_0}(\hat{\phi} = 1) + \mathbb{P}_{\mathbb{H}_1}(\hat{\phi} = 0)) \geq c$ for some constant $c > 0$.*

*Proof.* Note that $\phi$ is the likelihood ratio test. By the Neyman-Pearson lemma, it is the optimal procedure. That is, $\inf_{\hat{\phi}}(\mathbb{P}_{\mathbb{H}_0}(\hat{\phi} = 1) + \mathbb{P}_{\mathbb{H}_1}(\hat{\phi} = 0)) = \mathbb{P}_{\mathbb{H}_0}(\phi = 1) + \mathbb{P}_{\mathbb{H}_1}(\phi = 0)$. Let $\epsilon \sim \mathcal{N}(0, I_d)$. By Gaussian tail probability, we have

$$
\begin{aligned}
\mathbb{P}_{\mathbb{H}_0}(\phi = 1) + \mathbb{P}_{\mathbb{H}_1}(\phi = 0) &= \mathbb{P}\left(2(\theta_2^* - \theta_1^*)^T(\Sigma^*)^{-1}(\theta_1^* + \epsilon) \geq \theta_2^{*T}(\Sigma^*)^{-1}\theta_2^* - \theta_1^{*T}(\Sigma^*)^{-1}\theta_1^*\right) \\
&\quad + \mathbb{P}\left(2(\theta_2^* - \theta_1^*)^T(\Sigma^*)^{-1}(\theta_2^* + \epsilon) < \theta_2^{*T}(\Sigma^*)^{-1}\theta_2^* - \theta_1^{*T}(\Sigma^*)^{-1}\theta_1^*\right) \\
&= 2\mathbb{P}\left(2(\theta_2^* - \theta_1^*)^T(\Sigma^*)^{-1}(\theta_1^* + \epsilon) \geq \theta_2^{*T}(\Sigma^*)^{-1}\theta_2^* - \theta_1^{*T}(\Sigma^*)^{-1}\theta_1^*\right) \\
&= 2\mathbb{P}\left(\epsilon > \frac{1}{2}\|(\theta_2^* - \theta_1^*)^T(\Sigma^*)^{-\frac{1}{2}}\|\right) \\
&\geq C\min\left\{1, \frac{1}{\|(\theta_2^* - \theta_1^*)^T(\Sigma^*)^{-\frac{1}{2}}\|}\exp\left(-\frac{\|(\theta_2^* - \theta_1^*)^T(\Sigma^*)^{-\frac{1}{2}}\|^2}{8}\right)\right\},
\end{aligned}
$$

for some constant $C > 0$. The proof is complete. $\square$

*Proof of Theorem 2.1.* We adopt the idea from [15]. Without loss of generality, assume the minimum in (2) is achieved at $a = 1, b = 2$ so that SNR $= (\theta_1^* - \theta_2^*)^T(\Sigma^*)^{-1}(\theta_1^* - \theta_2^*)$. Consider an arbitrary $\bar{z} \in [k]^n$ such that $|\{i \in [n] : \bar{z}_i = a\}| \geq \lceil \frac{n}{k} - \frac{n}{8k^2} \rceil$ for any $a \in [k]$. Then for each $a \in [k]$, we can choose a subset of $\{i \in [n] : \bar{z}_i = a\}$ with cardinality $\lceil \frac{n}{k} - \frac{n}{8k^2} \rceil$, denoted by $T_a$. Let $T = \cup_{a \in [k]}T_a$. Then we can define a parameter space

$$\mathcal{Z} = \left\{z \in [k]^n : z_i = \bar{z}_i \text{ for all } i \in T \text{ and } z_i \in \{1, 2\} \text{ if } i \in T^c\right\}.$$

Notice that for any $z \neq \tilde{z} \in \mathcal{Z}$, we have $\frac{1}{n}\sum_{i=1}^n \mathbb{I}\{z_i \neq \tilde{z}_i\} \leq \frac{k}{n}\frac{n}{8k^2} = \frac{1}{8k}$ and $\frac{1}{n}\sum_{i=1}^n \mathbb{I}\{\psi(z_i) \neq \tilde{z}_i\} \geq \frac{1}{n}(\frac{n}{2k} - \frac{n}{8k^2}) \geq \frac{1}{4k}$ for any permutation $\psi$ on $[k]$. Thus we can conclude

$$h(z, \tilde{z}) = \frac{1}{n}\sum_{i=1}^n \mathbb{I}\{z_i \neq \tilde{z}_i\}, \quad \text{for all } z, \tilde{z} \in \mathcal{Z}.$$

We notice that

$$\inf_{\hat{z}} \sup_{z^* \in [k]^n} \mathbb{E}h(\hat{z}, z^*) \geq \inf_{\hat{z}} \sup_{z^* \in \mathcal{Z}} \mathbb{E}h(\hat{z}, z^*)$$

$$\geq \inf_{\hat{z}} \frac{1}{|\mathcal{Z}|} \sum_{z^* \in \mathcal{Z}} \mathbb{E}h(\hat{z}, z^*)$$

$$\geq \frac{1}{n} \sum_{i \in T^c} \inf_{\hat{z}_i} \frac{1}{|\mathcal{Z}|} \sum_{z^* \in \mathcal{Z}} \mathbb{P}_{z^*}(\hat{z}_i \neq z_i).$$

Now consider a fixed $i \in T^c$. Define $\mathcal{Z}_a = \{z \in \mathcal{Z} : z_i = a\}$ for $a = 1, 2$. Then we can see $\mathcal{Z} = \mathcal{Z}_1 \cup \mathcal{Z}_2$ and $\mathcal{Z}_1 \cap \mathcal{Z}_2 = \emptyset$. What is more, there exists a one-to-one mapping $f(\cdot)$ between $\mathcal{Z}_1$ and $\mathcal{Z}_2$, such that for any $z \in \mathcal{Z}_1$, we have $f(z) \in \mathcal{Z}_2$ with $[f(z)]_j = z_j$ for any $j \neq i$ and $[f(z)]_i = 2$. Hence, we can reduce the problem to a two-point testing probe and then apply Lemma A.1. We first consider the case that SNR $\to \infty$. We have

$$\inf_{\hat{z}_i} \frac{1}{|\mathcal{Z}|} \sum_{z^* \in \mathcal{Z}} \mathbb{P}_{z^*}(\hat{z}_i \neq z_i) = \inf_{\hat{z}_i} \frac{1}{|\mathcal{Z}|} \sum_{z^* \in \mathcal{Z}_1} \left( \mathbb{P}_{z^*}(\hat{z}_i \neq 1) + \mathbb{P}_{f(z^*)}(\hat{z}_i \neq 2) \right)$$

$$\geq \frac{1}{|\mathcal{Z}|} \sum_{z^* \in \mathcal{Z}_1} \inf_{\hat{z}_i} \left( \mathbb{P}_{z^*}(\hat{z}_i \neq 1) + \mathbb{P}_{f(z^*)}(\hat{z}_i \neq 2) \right)$$

$$\geq \frac{|\mathcal{Z}_1|}{\mathcal{Z}} \exp\left( -(1 + \eta)\frac{\text{SNR}^2}{8} \right)$$

$$\geq \frac{1}{2} \exp\left( -(1 + \eta)\frac{\text{SNR}^2}{8} \right),$$

for some $\eta = o(1)$. Here the second inequality is due to Lemma A.1. Then,

$$\inf_{\hat{z}} \sup_{z^* \in [k]^n} \mathbb{E}h(\hat{z}, z^*) \geq \frac{|T^c|}{2n} \exp\left( -(1 + \eta)\frac{\text{SNR}^2}{8} \right) = \frac{1}{16k} \exp\left( -(1 + \eta)\frac{\text{SNR}^2}{8} \right)$$

$$= \exp\left( -(1 + \eta')\frac{\text{SNR}^2}{8} \right),$$

for some other $\eta' = o(1)$, where we use $\text{SNR}^2 / \log k \to \infty$.

The proof for the case SNR $= O(1)$ is similar and hence is omitted here. $\qquad\square$

## A.2 Proofs for the Upper Bound

### A.2.1 High-level Idea

In this section, we provide a high-level idea for the proof of Theorem 2.2. The detailed proof is technical and is given later in Appendix A.2.2.

The key idea for establishing the statistical guarantees of Algorithm 1, an iterative algorithm, is to perform a "one-step" analysis [8]. That is, assume we have an estimation $z$ for $z^*$. Then we can apply (7), (8), and (9) on $z$ to obtain $\{\hat{\theta}_a(z)\}_{a \in [k]}$, $\hat{\Sigma}(z)$, and $\hat{z}(z)$ sequentially, which all depend on $z$. Thus, $\hat{z}(z)$ can be seen as a refined estimate of $z^*$. We will first build the connection between $\ell(z, z^*)$ with $\ell(\hat{z}(z), z^*)$ as in Lemma A.2, which informally states that under certain conditions, with high probability, we have

$$\ell(\hat{z}(z), z^*) \leq \xi_{\text{ideal}}(\delta) + \frac{1}{2}\ell(z, z^*)$$

holds for any $z \in [k]^n$ such that $\ell(z, z^*)$ is small. Here $\xi_{\text{ideal}}(\delta)$ refers to the ideal error, which eventually leads to the upper bound in Theorem 2.2. Lemma A.2 tells us $\hat{z}(\cdot)$ has a "contraction" property. That is, after one iteration of (7), (8), and (9), $\ell(\hat{z}(z), z^*)$ is at most a half of $\ell(z, z^*)$, up to an additive term $\xi_{\text{ideal}}(\delta)$.

To establish Lemma A.2, we decompose the loss $\ell(\hat{z}(z), z^*)$ into several errors according to the difference in their behaviors. Next, we will introduce several conditions (Conditions 1 - 3), under

which we demonstrate that these errors are either negligible, well-controlled by $\ell(z, z^*)$, or connected to $\xi_{\text{ideal}}(\delta)$. Once Lemma A.2 is established, we will show in Lemma A.3 that the connection can be extended to multiple iterations, under two more conditions (Conditions 4 - 5). Lemma A.3 states informally that, under certain conditions and with high probability, we have

$$\ell(z^{(t)}, z^*) \leq \xi_{\text{ideal}}(\delta) + \frac{1}{2}\ell(z^{(t-1)}, z^*)$$

for all $t \geq 1$. This implies $\ell(z^{(t)}, z^*)$ is eventually at most $\xi_{\text{ideal}}(\delta)$, up to some constant factor. Last, we will show all these conditions hold with high probability. Although the algorithmic guarantees in Lemma A.2 and Lemma A.3 are established with respect to the $\ell(\cdot, \cdot)$ loss, we will use the relationship between $h(\cdot, \cdot)$ and $\ell(\cdot, \cdot)$ to convert this result to one involving $h(\cdot, \cdot)$. Hence, we prove Theorem 2.2.

### A.2.2 Detailed Proofs

In the statement of Theorem 2.2, the covariance matrix $\Sigma^*$ is assumed to satisfy $\lambda_d(\Sigma^*)/\lambda_1(\Sigma^*) = O(1)$. Without loss of generality, we can replace it by assuming $\Sigma^*$ satisfies

$$\lambda_{\min} \leq \lambda_1(\Sigma^*) \leq \lambda_d(\Sigma^*) \leq \lambda_{\max} \tag{15}$$

where $\lambda_{\min}, \lambda_{\max} > 0$ are two constants. This is due to the following simple argument using the scaling properties of normal distributions. Let $\{Y_j\}$ be some dataset generated according to Model 1 with parameters $\{\theta_a^*\}_{a \in [k]}$, $\Sigma^*$, and $z^*$. The assumption $\lambda_d(\Sigma^*)/\lambda_1(\Sigma^*) = O(1)$ is equivalent to assuming there exist some constants $\lambda_{\min}, \lambda_{\max} > 0$ and some quantity $\sigma > 0$ that may depend on $n$ such that $\lambda_{\min}\sigma^2 \leq \lambda_1(\Sigma^*) \leq \lambda_d(\Sigma^*) \leq \lambda_{\max}\sigma^2$. By performing a scaling transformation, we obtain another dataset $Y_j' = Y_j/\sigma$. Note that: 1) $\{Y_j'\}$ can be seen as generated from Model 1 with parameters $\{\theta_a^*/\sigma\}_{a \in [k]}$, $\Sigma^*/\sigma^2$, and $z^*$. 2) Clustering on $\{Y_j\}$ is equivalent to clustering on $\{Y_j'\}$. 3) By the definition in (2), the SNRs that are associated with the data-generating processes of $\{Y_j'\}$ and $\{Y_j\}$ are exactly equal to each other. 4) We have $\lambda_{\min} \leq \lambda_1(\Sigma^*/\sigma^2) \leq \lambda_d(\Sigma^*/\sigma^2) \leq \lambda_{\max}$. Thus, for the remainder of this section, we assume that (15) holds without any loss of generality.

In the proof, we will mainly use the loss $\ell(\cdot, \cdot)$ for convenience. Recall $\Delta$ is defined as the minimum distance among centers in (3). We have

$$h(z, z^*) \leq \frac{\ell(z, z^*)}{n\Delta^2}. \tag{16}$$

The algorithmic guarantees Lemma A.2 and Lemma A.3 are established with respect to the $\ell(\cdot, \cdot)$ loss. Eventually, we will use (16) to convert it into a result with respect to $h(\cdot, \cdot)$ in the proof of Theorem 2.2.

**Error Decomposition for the One-step Analysis:** Consider an arbitrary $z \in [k]^n$. Apply (7), (8), and (9) on $z$ to obtain $\{\hat{\theta}_a(z)\}_{a \in [k]}$, $\hat{\Sigma}(z)$, and $\hat{z}(z)$:

$$\hat{\theta}_a(z) = \frac{\sum_{j \in [n]} Y_j \mathbb{I}\{z_j = a\}}{\sum_{j \in [n]} \mathbb{I}\{z_j = a\}}, \quad \forall a \in [k]$$

$$\hat{\Sigma}(z) = \frac{\sum_{a \in [k]} \sum_{j \in [n]} (Y_j - \hat{\theta}_a(z))(Y_j - \hat{\theta}_a(z))^T \mathbb{I}\{z_j = a\}}{n},$$

$$\hat{z}_j(z) = \operatorname*{argmin}_{a \in [k]} (Y_j - \hat{\theta}_a(z))^T (\hat{\Sigma}(z))^{-1}(Y_j - \hat{\theta}_a), \quad \forall j \in [n].$$

For simplicity, we denote $\hat{z}$ as shorthand for $\hat{z}(z)$. Let $j \in [n]$ be an arbitrary index with $z_j^* = a$. According to (9), $z_j^*$ will be incorrectly estimated after one iteration in $\hat{z}$ if $a \neq \operatorname{argmin}_{b \in [k]}(Y_j - \hat{\theta}_b(z))^T(\hat{\Sigma}(z))^{-1}(Y_j - \hat{\theta}_b(z))$. Therefore, it is important to analyze the event

$$\langle Y_j - \hat{\theta}_b(z), (\hat{\Sigma}(z))^{-1}(Y_j - \hat{\theta}_b(z))\rangle \leq \langle Y_j - \hat{\theta}_a(z), (\hat{\Sigma}(z))^{-1}(Y_j - \hat{\theta}_a(z))\rangle, \tag{17}$$

for any $b \in [k] \setminus \{a\}$. Note that $Y_j = \theta_a^* + \epsilon_j$. After some rearrangements, we can see (17) is equivalent to

$$\langle \epsilon_j, (\hat{\Sigma}(z^*))^{-1}(\hat{\theta}_a(z^*) - \hat{\theta}_b(z^*))\rangle$$
$$\leq -\frac{1}{2}\langle \theta_a^* - \theta_b^*, (\Sigma^*)^{-1}(\theta_a^* - \theta_b^*)\rangle + F_j(a, b, z) + G_j(a, b, z) + H_j(a, b, z),$$

where

$$F_j(a, b, z) = \langle \epsilon_j, (\hat{\Sigma}(z))^{-1}(\hat{\theta}_b(z) - \hat{\theta}_b(z^*)) \rangle - \langle \epsilon_j, (\hat{\Sigma}(z))^{-1}(\hat{\theta}_a(z) - \hat{\theta}_a(z^*)) \rangle$$
$$+ \langle \epsilon_j, ((\hat{\Sigma}(z))^{-1} - (\hat{\Sigma}(z^*))^{-1})(\hat{\theta}_b(z^*) - \hat{\theta}_a(z^*)) \rangle,$$

$$G_j(a, b, z) = \frac{1}{2}\langle \theta_a^* - \hat{\theta}_a(z), (\hat{\Sigma}(z))^{-1}(\theta_a^* - \hat{\theta}_a(z)) \rangle - \frac{1}{2}\langle \theta_a^* - \hat{\theta}_a(z^*), (\hat{\Sigma}(z))^{-1}(\theta_a^* - \hat{\theta}_a(z^*)) \rangle$$
$$+ \frac{1}{2}\langle \theta_a^* - \hat{\theta}_a(z^*), (\hat{\Sigma}(z))^{-1}(\theta_a^* - \hat{\theta}_a(z^*)) \rangle - \frac{1}{2}\langle \theta_a^* - \hat{\theta}_a(z^*), (\hat{\Sigma}(z^*))^{-1}(\theta_a^* - \hat{\theta}_a(z^*)) \rangle$$
$$- \frac{1}{2}\langle \theta_a^* - \hat{\theta}_b(z), (\hat{\Sigma}(z))^{-1}(\theta_a^* - \hat{\theta}_b(z)) \rangle + \frac{1}{2}\langle \theta_a^* - \hat{\theta}_b(z^*), (\hat{\Sigma}(z))^{-1}(\theta_a^* - \hat{\theta}_b(z^*)) \rangle$$
$$- \frac{1}{2}\langle \theta_a^* - \hat{\theta}_b(z^*), (\hat{\Sigma}(z))^{-1}(\theta_a^* - \hat{\theta}_b(z^*)) \rangle + \frac{1}{2}\langle \theta_a^* - \hat{\theta}_b(z^*), (\hat{\Sigma}(z^*))^{-1}(\theta_a^* - \hat{\theta}_b(z^*)) \rangle,$$

$$H_j(a, b, z) = -\frac{1}{2}\langle \theta_a^* - \hat{\theta}_b(z^*), (\hat{\Sigma}(z^*))^{-1}(\theta_a^* - \hat{\theta}_b(z^*)) \rangle + \frac{1}{2}\langle \theta_a^* - \theta_b^*, (\hat{\Sigma}(z^*))^{-1}(\theta_a^* - \theta_b^*) \rangle$$
$$- \frac{1}{2}\langle \theta_a^* - \theta_b^*, (\hat{\Sigma}(z^*))^{-1}(\theta_a^* - \theta_b^*) \rangle + \frac{1}{2}\langle \theta_a^* - \theta_b^*, (\Sigma^*)^{-1}(\theta_a^* - \theta_b^*) \rangle$$
$$+ \frac{1}{2}\langle \theta_a^* - \hat{\theta}_a(z^*), (\hat{\Sigma}(z^*))^{-1}(\theta_a^* - \hat{\theta}_a(z^*)) \rangle.$$

In the above decomposition, the expression $\langle \epsilon_j, (\hat{\Sigma}(z^*))^{-1}(\hat{\theta}_a(z^*) - \hat{\theta}_b(z^*)) \rangle \le -\frac{1}{2}\langle \theta_a^* - \theta_b^*, (\Sigma^*)^{-1}(\theta_a^* - \theta_b^*) \rangle$ does not involve $z$. Roughly speaking, it corresponds to the event that $z_j^*$ will be incorrectly estimated in $\hat{z}(z^*)$. This is considered the main part of (17) and will contribute to $\xi_{\text{ideal}}$. The difference between (17) and the main term is expressed through the terms $F_j, G_j, H_j$: $F_j$ includes terms related to noise $\epsilon_j$, illustrating the impact of measurement noise; $G_j$ covers estimation errors for cluster centers $(\hat{\theta}_a(z) - \hat{\theta}_a(z^*))$ and covariance matrices $(\hat{\Sigma}(z) - \hat{\Sigma}(z^*))$, showing the effect of the parameter estimation inaccuracies; $H_j$ contains all other terms from additional error sources. Readers can refer to [8] for more information about the decomposition.

**Conditions and Guarantees for One-step Analysis.** We continue to analyze the event (17). We first define a quantity independent of $z$, which we refer to as the ideal error:

$$\xi_{\text{ideal}}(\delta) = \sum_{j=1}^{n} \sum_{b \in [k] \setminus \{z_j^*\}} \|\theta_{z_j^*}^* - \theta_b^*\|^2 \mathbb{I}\Bigg\{ \langle \epsilon_j, (\hat{\Sigma}(z^*))^{-1}(\hat{\theta}_a(z^*) - \hat{\theta}_b(z^*)) \rangle$$
$$\le -\frac{1 - \delta}{2}\langle \theta_a^* - \theta_b^*, (\Sigma^*)^{-1}(\theta_a^* - \theta_b^*) \rangle \Bigg\}.$$

When $\delta = 0$, it is determined by the main term in (17), namely $\langle \epsilon_j, (\hat{\Sigma}(z^*))^{-1}(\hat{\theta}_a(z^*) - \hat{\theta}_b(z^*)) \rangle \le -\frac{1}{2}\langle \theta_a^* - \theta_b^*, (\Sigma^*)^{-1}(\theta_a^* - \theta_b^*) \rangle$. Roughly speaking, $\xi_{\text{ideal}}(0)$ relates to the performance of $\hat{z}(z^*)$. Due to the presence of the terms $F_j, G_j, H_j$ in the decomposition of (17), what appears in the analysis of (17) is $\xi_{\text{ideal}}(\delta)$ instead of $\xi_{\text{ideal}}(0)$ where hopefully $\delta > 0$ is some small number.

To establish the guarantee for one-step analysis, we next give several conditions on the error terms $F_j(a, b; z), G_j(a, b; z)$ and $H_j(a, b; z)$.

**Condition 1.** *Assume that*

$$\max_{\{z: l(z, z^*) \le \tau\}} \max_{j \in [n]} \max_{b \in [k] \setminus \{z_j^*\}} \frac{|H_j(z_j^*, b, z)|}{\langle \theta_{z_j^*}^* - \theta_b^*, (\Sigma^*)^{-1}(\theta_{z_j^*}^* - \theta_b^*) \rangle} \le \frac{\delta}{4}$$

*holds with probability at least $1 - \eta_1$ for some $\tau, \delta, \eta_1 > 0$.*

**Condition 2.** *Assume that*

$$\max_{\{z: l(z, z^*) \le \tau\}} \sum_{j=1}^{n} \max_{b \in [k] \setminus \{z_j^*\}} \frac{F_j(z_j^*, b, z)^2 \|\theta_{z_j^*}^* - \theta_b^*\|^2}{\langle \theta_{z_j^*}^* - \theta_b^*, (\Sigma^*)^{-1}(\theta_{z_j^*}^* - \theta_b^*) \rangle^2 \ell(z, z^*)} \le \frac{\delta^2}{128}$$

*holds with probability at least $1 - \eta_2$ for some $\tau, \delta, \eta_2 > 0$.*

**Condition 3.** *Assume that*

$$\max_{\{z:l(z,z^*)\leq\tau\}}\max_{j\in[n]}\max_{b\in[k]\backslash\{z_j^*\}}\frac{|G_j(z_j^*,b,z)|}{\langle\theta_{z_j^*}^*-\theta_b^*,(\Sigma^*)^{-1}(\theta_{z_j^*}^*-\theta_b^*)\rangle}\leq\frac{\delta}{8}$$

*holds with probability at least $1-\eta_3$ for some $\tau,\delta,\eta_3>0$.*

**Lemma A.2.** *Assumes Conditions 1 - 3 hold for some $\tau,\delta,\eta_1,\eta_2,\eta_3,>0$. We then have*

$$\mathbb{P}\left(\ell(\hat{z},z^*)\leq\xi_{ideal}(\delta)+\frac{1}{2}\ell(z,z^*)\text{ for any }z\in[k]^n\text{ such that }\ell(z,z^*)\leq\tau\right)\geq1-\eta,$$

*where $\eta=\sum_{i=1}^{3}\eta_i$.*

*Proof.* Consider any $j\in[n]$ such that $z_j^*=a$. We notice that for any $b\in[k]$ such that $b\neq a$,

$$\mathbb{I}\{\hat{z}_j=b\}\leq\mathbb{I}\left\{\langle Y_j-\hat{\theta}_b(z),(\hat{\Sigma}(z))^{-1}(Y_j-\hat{\theta}_b(z))\rangle\leq\langle Y_j-\hat{\theta}_a(z),(\hat{\Sigma}(z))^{-1}(Y_j-\hat{\theta}_a(z))\rangle\right\}$$

$$=\mathbb{I}\Bigg\{\langle\epsilon_j,(\hat{\Sigma}(z^*))^{-1}(\hat{\theta}_{z_j^*}(z^*)-\hat{\theta}_b(z^*))\rangle$$

$$\leq-\frac{1}{2}\langle\theta_{z_j^*}^*-\theta_b^*,(\Sigma^*)^{-1}(\theta_{z_j^*}^*-\theta_b^*)\rangle+F_j(z_j^*,b,z)+G_j(z_j^*,b,z)+H_j(z_j^*,b,z)\Bigg\}$$

$$\leq\mathbb{I}\left\{\langle\epsilon_j,(\hat{\Sigma}(z^*))^{-1}(\hat{\theta}_{z_j^*}(z^*)-\hat{\theta}_b(z^*))\rangle\leq-\frac{1-\delta}{2}\langle\theta_{z_j^*}^*-\theta_b^*,(\Sigma^*)^{-1}(\theta_{z_j^*}^*-\theta_b^*)\rangle\right\}$$

$$+\mathbb{I}\left\{\frac{\delta}{2}\langle\theta_{z_j^*}^*-\theta_b^*,(\Sigma^*)^{-1}(\theta_{z_j^*}^*-\theta_b^*)\rangle\leq F_j(z_j^*,b,z)+G_j(z_j^*,b,z)+H_j(z_j^*,b,z)\right\}$$

$$\leq\mathbb{I}\left\{\langle\epsilon_j,(\hat{\Sigma}(z^*))^{-1}(\hat{\theta}_{z_j^*}(z^*)-\hat{\theta}_b(z^*))\rangle\leq-\frac{1-\delta}{2}\langle\theta_{z_j^*}^*-\theta_b^*,(\Sigma^*)^{-1}(\theta_{z_j^*}^*-\theta_b^*)\rangle\right\}$$

$$+\mathbb{I}\left\{\frac{\delta}{8}\langle\theta_{z_j^*}^*-\theta_b^*,(\Sigma^*)^{-1}(\theta_{z_j^*}^*-\theta_b^*)\rangle\leq F_j(z_j^*,b,z)\right\}$$

$$\leq\mathbb{I}\left\{\langle\epsilon_j,(\hat{\Sigma}(z^*))^{-1}(\hat{\theta}_{z_j^*}(z^*)-\hat{\theta}_b(z^*))\rangle\leq-\frac{1-\delta}{2}\langle\theta_{z_j^*}^*-\theta_b^*,(\Sigma^*)^{-1}(\theta_{z_j^*}^*-\theta_b^*)\rangle\right\}$$

$$+\frac{64F_j(z_j^*,b,z)^2}{\delta^2\langle\theta_{z_j^*}^*-\theta_b^*,(\Sigma^*)^{-1}(\theta_{z_j^*}^*-\theta_b^*)\rangle^2},$$

where the second inequality comes from Conditions 1 and 3. Note that we can multiply $\mathbb{I}\{\hat{z}_j=b\}$ on both sides of the above display and the inequality still holds. Hence,

$$\mathbb{I}\{\hat{z}_j=b\}\leq\mathbb{I}\left\{\langle\epsilon_j,(\hat{\Sigma}(z^*))^{-1}(\hat{\theta}_{z_j^*}(z^*)-\hat{\theta}_b(z^*))\rangle\leq-\frac{1-\delta}{2}\langle\theta_{z_j^*}^*-\theta_b^*,(\Sigma^*)^{-1}(\theta_{z_j^*}^*-\theta_b^*)\rangle\right\}$$

$$+\frac{64F_j(z_j^*,b,z)^2}{\delta^2\langle\theta_{z_j^*}^*-\theta_b^*,(\Sigma^*)^{-1}(\theta_{z_j^*}^*-\theta_b^*)\rangle^2}\mathbb{I}\{\hat{z}_j=b\}.$$

Thus, we have

$$\ell(\hat{z},z^*)$$

$$=\sum_{j=1}^{n}\sum_{b\in[k]\backslash\{a\}}\left\|\theta_b^*-\theta_{z_j^*}^*\right\|^2\mathbb{I}\{\hat{z}_j=b\}$$

$$\leq\xi_{ideal}(\delta)+\sum_{j=1}^{n}\sum_{b\in[k]\backslash\{z_j^*\}}\left\|\theta_b^*-\theta_{z_j^*}^*\right\|^2\mathbb{I}\{\hat{z}_j=b\}\frac{64F_j(z_j^*,b,z)^2}{\delta^2\langle\theta_{z_j^*}^*-\theta_b^*,(\Sigma^*)^{-1}(\theta_{z_j^*}^*-\theta_b^*)\rangle^2}$$

$$\leq\xi_{ideal}(\delta)+\sum_{j=1}^{n}\max_{b\in[k]\backslash\{z_j^*\}}\left\|\theta_b^*-\theta_{z_j^*}^*\right\|^2\frac{64F_j(z_j^*,b,z)^2}{\delta^2\langle\theta_{z_j^*}^*-\theta_b^*,(\Sigma^*)^{-1}(\theta_{z_j^*}^*-\theta_b^*)\rangle^2}$$

$$\leq\xi_{ideal}(\delta)+\frac{\ell(z,z^*)}{2},$$

which implies Lemma A.2. Here the last inequality uses Condition 2. $\qquad\square$

**Conditions and Guarantees for Multiple Iterations.** In the above, we establish a statistical guarantee for the one-step analysis. Now we will extend the result to multiple iterations. That is, starting from some initialization $z^{(0)}$, we will characterize how the losses $\ell(z^{(0)}, z^*)$, $\ell(z^{(1)}, z^*)$, $\ell(z^{(2)}, z^*)$, ..., decay. We impose conditions on $\xi_{\text{ideal}}(\delta)$ and the initialization $z^{(0)}$.

**Condition 4.** *Assume that*

$$\xi_{ideal}(\delta) \leq \frac{3\tau}{8}$$

*holds with probability at least $1 - \eta_4$ for some $\tau, \delta, \eta_4 > 0$.*

Finally, we need a condition on the initialization.

**Condition 5.** *Assume that*

$$\ell(z^{(0)}, z^*) \leq \tau$$

*holds with probability at least $1 - \eta_5$ for some $\tau, \eta_5 > 0$.*

With these conditions satisfied, we can give a lemma that shows the convergence of our algorithm.

**Lemma A.3.** *Assume Conditions 1 - 5 hold for some $\tau, \delta, \eta_1, \eta_2, \eta_3, \eta_4, \eta_5 > 0$. We then have*

$$\ell(z^{(t)}, z^*) \leq \xi_{ideal}(\delta) + \frac{1}{2}\ell(z^{(t-1)}, z^*)$$

*for all $t \geq 1$, with probability at least $1 - \eta$, where $\eta = \sum_{i=1}^{5} \eta_i$.*

*Proof.* By Conditions 4, 5 and a mathematical induction argument, we can easily conclude $\ell(z^{(t)}, z^*) \leq \tau$ for any $t \geq 0$. Thus, Lemma A.3 is a direct extension of Lemma A.2. $\square$

**With-high-probability Results for the Conditions and Proof of Theorem 2.2.** Recall the definition of $\Delta$ in (3). Recall that in (15) we assume $\lambda_{\min} \leq \lambda_1(\Sigma^*) \leq \lambda_d(\Sigma^*) \leq \lambda_{\max}$ for two constants $\lambda_{\min}, \lambda_{\max} > 0$. Hence we have $\Delta$ is of the same order as SNR. Specifically, we have

$$\frac{1}{\sqrt{\lambda_{\max}}}\Delta \leq \text{SNR} \leq \frac{1}{\sqrt{\lambda_{\min}}}\Delta. \tag{18}$$

Hence the assumption SNR $\to \infty$ in the statement of Theorem 2.2 is equivalently $\Delta \to \infty$. Next, we give two with-high-probability lemmas. The first lemma is for Conditions 1-3, providing upper bounds for the quantities involved in these conditions, showing that $\delta$ can be taken as some $o(1)$ term. The second lemma shows that for any $\delta = o(1)$, $\xi_{\text{ideal}}(\delta)$ is upper bounded by the desired minimax rate multiplied by the sample size $n$.

**Lemma A.4.** *Under the same conditions as in Theorem 2.2, for any constant $C' > 0$, there exists some constant $C > 0$ only depending on $\alpha$ and $C'$ such that*

$$\max_{\{z:\ell(z,z^*)\leq\tau\}} \max_{j\in[n]} \max_{b\in[k]\backslash\{z_j^*\}} \frac{|H_j(z_j^*, b, z)|}{\langle\theta_{z_j^*}^* - \theta_b^*, (\Sigma^*)^{-1}(\theta_{z_j^*}^* - \theta_b^*)\rangle} \leq C\sqrt{\frac{k(d + \log n)}{n}} \tag{19}$$

$$\max_{\{z:\ell(z,z^*)\leq\tau\}} \sum_{j=1}^{n} \max_{b\in[k]\backslash\{z_j^*\}} \frac{F_j(z_j^*, b, z)^2\|\theta_{z_j^*}^* - \theta_b^*\|^2}{\langle\theta_{z_j^*}^* - \theta_b^*, (\Sigma^*)^{-1}(\theta_{z_j^*}^* - \theta_b^*)\rangle^2\ell(z,z^*)} \leq Ck^3\left(\frac{\tau}{n} + \frac{1}{\Delta^2} + \frac{d^2}{n\Delta^2}\right) \tag{20}$$

$$\max_{\{z:\ell(z,z^*)\leq\tau\}} \max_{j\in[n]} \max_{b\in[k]\backslash\{z_j^*\}} \frac{|G_j(z_j^*, b, z)|}{\langle\theta_{z_j^*}^* - \theta_b^*, (\Sigma^*)^{-1}(\theta_{z_j^*}^* - \theta_b^*)\rangle} \leq Ck\left(\frac{\tau}{n} + \frac{1}{\Delta}\sqrt{\frac{\tau}{n}} + \frac{d\sqrt{\tau}}{n\Delta}\right) \tag{21}$$

*with probability at least $1 - n^{-C'}$. As a result, Conditions 1-3 hold for some $\delta = o(1)$.*

*Proof.* Under the conditions of Theorem 2.2, the inequalities (33)-(38) hold with probability at least $1 - n^{-C'}$. In the remaining proof, we will work on the event these inequalities hold. Denote

$\hat{\Sigma}_a(z) = \frac{\sum_{j\in[n]}(Y_j-\hat{\theta}_a(z))(Y_j-\hat{\theta}_a(z))^T\mathbb{I}\{z_j=a\}}{\sum_{j\in[n]}\mathbb{I}\{z_j=a\}}$ and $\Sigma_a^* = \Sigma^*$ for any $a\in[k]$. Then we have the equivalence

$$\hat{\Sigma}(z^*) - \Sigma^* = \sum_{a=1}^k \frac{\sum_{j=1}^n \mathbb{I}\{z_j^*=a\}}{n}(\hat{\Sigma}_a(z^*) - \Sigma_a^*).$$

Hence, we can use the results from Lemma C.7 and Lemma C.8.

By (43) and (44), we have

$$\|\hat{\Sigma}(z^*) - \Sigma^*\| \preceq \sqrt{\frac{k(d+\log n)}{n}},$$

and

$$\|\hat{\Sigma}(z) - \hat{\Sigma}(z^*)\| = \left\| \sum_{a=1}^k \frac{\sum_{j=1}^n \mathbb{I}\{z_j=a\}}{n}\hat{\Sigma}_a(z) - \sum_{a=1}^k \frac{\sum_{j=1}^n \mathbb{I}\{z_j^*=a\}}{n}\hat{\Sigma}_a(z^*) \right\|$$

$$\preceq \left\| \sum_{a=1}^k \frac{\sum_{j=1}^n \mathbb{I}\{z_j=a\}}{n}(\hat{\Sigma}_a(z) - \hat{\Sigma}(z^*)) \right\| + \left\| \sum_{a=1}^k \frac{\sum_{j=1}^n (\mathbb{I}\{z_j=a\} - \mathbb{I}\{z_j^*=a\})}{n}\hat{\Sigma}_a(z^*) \right\|$$

$$\preceq \frac{k\sqrt{n\ell(z,z^*)}}{n\Delta} + \frac{k}{n}\ell(z,z^*) + \frac{kd}{n\Delta}\sqrt{\ell(z,z^*)} + \frac{k}{n\Delta^2}\ell(z,z^*)$$

$$\preceq \frac{k\sqrt{n\ell(z,z^*)}}{n\Delta} + \frac{k}{n}\ell(z,z^*) + \frac{kd}{n\Delta}\sqrt{\ell(z,z^*)}.$$

By the assumption that $kd=O(\sqrt{n})$, $\frac{\Delta}{k}\to\infty$ and $\tau=o(n/k)$, we have $\|\hat{\Sigma}(z^*)-\Sigma^*\|, \|\hat{\Sigma}(z) - \hat{\Sigma}(z^*)\| = o(1)$, which implies $\|(\hat{\Sigma}(z^*))^{-1}\|, \|(\hat{\Sigma}(z))^{-1}\| \preceq 1$. Thus, we have

$$\|(\hat{\Sigma}(z^*))^{-1} - (\Sigma^*)^{-1}\| \le \|(\hat{\Sigma}(z^*))^{-1}\|\|\hat{\Sigma}(z^*) - \Sigma^*\|\|(\Sigma^*)^{-1}\| \preceq \sqrt{\frac{k(d+\log n)}{n}}, \qquad (22)$$

and similarly

$$\|(\hat{\Sigma}(z))^{-1} - (\hat{\Sigma}(z^*))^{-1}\| \preceq \frac{k}{n}\ell(z,z^*) + \frac{k\sqrt{n\ell(z,z^*)}}{n\Delta} + \frac{kd}{n\Delta}\sqrt{\ell(z,z^*)}. \qquad (23)$$

Now we start to prove (19)-(21). Let $F_j(a,b,z) = F_j^{(1)}(a,b,z) + F_j^{(2)}(a,b,z) + F_j^{(3)}(a,b,z)$ where

$F_j^{(1)}(a,b,z) := \langle \epsilon_j, (\hat{\Sigma}(z))^{-1}(\hat{\theta}_b(z) - \hat{\theta}_b(z^*))\rangle - \langle \epsilon_j, (\hat{\Sigma}(z))^{-1}(\hat{\theta}_a(z) - \hat{\theta}_a(z^*))\rangle$,

$F_j^{(2)}(a,b,z) := -\langle \epsilon_j, ((\hat{\Sigma}(z))^{-1} - (\hat{\Sigma}(z^*))^{-1})(\theta_a^* - \theta_b^*)\rangle$,

$F_j^{(3)}(a,b,z) := -\langle \epsilon_j, ((\hat{\Sigma}(z))^{-1} - (\hat{\Sigma}(z^*))^{-1})(\theta_b^* - \hat{\theta}_b(z^*))\rangle + \langle \epsilon_j, ((\hat{\Sigma}(z))^{-1} - (\hat{\Sigma}(z^*))^{-1})(\theta_a^* - \hat{\theta}_a(z^*))\rangle$.

Notice that

$$\sum_{j=1}^n \max_{b\in[k]\setminus\{z_j^*\}} \frac{F_j^{(2)}(z_j^*,b,z)^2\|\theta_{z_j^*}^* - \theta_b^*\|^2}{\langle \theta_{z_j^*}^* - \theta_b^*, (\Sigma^*)^{-1}(\theta_{z_j^*}^* - \theta_b^*)\rangle^2 \ell(z,z^*)}$$

$$\preceq \sum_{j=1}^n \sum_{b=1}^k \frac{\left|\langle \epsilon_j, ((\hat{\Sigma}(z))^{-1} - (\hat{\Sigma}(z^*))^{-1})(\theta_{z_j^*}^* - \theta_b^*)\rangle\right|^2}{\|\theta_{z_j^*}^* - \theta_b^*\|^2 \ell(z,z^*)}$$

$$\le \sum_{b=1}^k \sum_{a\in[k]\setminus\{b\}} \sum_{j=1}^n \mathbb{I}\{z_j^*=a\} \frac{\left|\langle \epsilon_j, ((\hat{\Sigma}(z))^{-1} - (\hat{\Sigma}(z^*))^{-1})(\theta_a^* - \theta_b^*)\rangle\right|^2}{\|\theta_a^* - \theta_b^*\|^2 \ell(z,z^*)}$$

$$\le \sum_{b=1}^k \sum_{a\in[k]\setminus\{b\}} \frac{\|((\hat{\Sigma}(z))^{-1} - (\hat{\Sigma}(z^*))^{-1})(\theta_a^* - \theta_b^*)\|^2}{\|\theta_a^* - \theta_b^*\|^2 \ell(z,z^*)} \left\| \sum_{j=1}^n \mathbb{I}\{z_j^*=a\}\epsilon_j\epsilon_j^T \right\|$$

$$\preceq k^3(\frac{\tau}{n} + \frac{1}{\Delta^2} + \frac{d^2}{n\Delta^2}),$$

where we use (34), (23), and the fact that $\ell(z, z^*) \leq \tau$ and $kd = O(\sqrt{n})$ for the last inequality. Here the second to last inequality is due to the following argument: for any $w \in \mathbb{R}^d$, we have $\sum_j |\langle \epsilon_j, w \rangle|^2 = \sum_j w^T \epsilon_j \epsilon_j^T w = w^T (\sum_j \epsilon_j \epsilon_j^T) w \leq \|w\|^2 \| \sum_j \epsilon_j \epsilon_j^T \|$. From (41) we have $\max_{a \in [k]} \|\theta_a^* - \hat{\theta}_a(z^*)\| = o(1)$ under the assumption $kd = O(\sqrt{n})$. By the similar analysis as in $F_j^{(2)}(a, b, z)$, we have

$$\sum_{j=1}^n \max_{b \in [k] \setminus \{z_j^*\}} \frac{F_j^{(3)}(z_j^*, b, z)^2 \|\theta_{z_j^*}^* - \theta_b^*\|^2}{\langle \theta_{z_j^*}^* - \theta_b^*, (\Sigma^*)^{-1}(\theta_{z_j^*}^* - \theta_b^*) \rangle^2 \ell(z, z^*)} \preceq k^3 \left( \frac{\tau}{n} + \frac{1}{\Delta^2} + \frac{d^2}{n\Delta^2} \right).$$

Similarly, we have

$$\sum_{j=1}^n \max_{b \in [k] \setminus \{z_j^*\}} \frac{F_j^{(1)}(z_j^*, b, z)^2 \|\theta_{z_j^*}^* - \theta_b^*\|^2}{\langle \theta_{z_j^*}^* - \theta_b^*, (\Sigma^*)^{-1}(\theta_{z_j^*}^* - \theta_b^*) \rangle^2 \ell(z, z^*)}$$

$$\preceq \sum_{b=1}^k \sum_{a \in [k] \setminus \{b\}} \frac{\|(\hat{\Sigma}(z))^{-1}(\hat{\theta}_a(z) - \hat{\theta}_a(z^*))\|^2}{\|\theta_a^* - \theta_b^*\|^2 \ell(z, z^*)} \left\| \sum_{j=1}^n \mathbb{I}\{z_j^* = a\} \epsilon_j \epsilon_j^T \right\|$$

$$\preceq \frac{k^3}{\Delta^4},$$

where we use (42) and the fact that $(\hat{\Sigma}(z))^{-1}$ has bounded operator norm. Combining these terms together, we obtain (20).

Next, for (19), by (41) we have

$$| -\langle \theta_a^* - \hat{\theta}_b(z^*), (\hat{\Sigma}(z^*))^{-1}(\theta_a^* - \hat{\theta}_b(z^*)) \rangle + \langle \theta_a^* - \theta_b^*, (\hat{\Sigma}(z^*))^{-1}(\theta_a^* - \theta_b^*) \rangle |$$

$$\leq |\langle \theta_b^* - \hat{\theta}_b(z^*), (\hat{\Sigma}(z^*))^{-1}(\theta_b^* - \hat{\theta}_b(z^*)) \rangle| + 2|\langle \theta_b^* - \hat{\theta}_b(z^*), (\hat{\Sigma}(z^*))^{-1}(\theta_a^* - \theta_b^*) \rangle|$$

$$\preceq \frac{k(d + \log n)}{n} + \sqrt{\frac{k(d + \log n)}{n}} \|\theta_a^* - \theta_b^*\|,$$

and

$$|\langle \theta_a^* - \hat{\theta}_a(z^*), (\hat{\Sigma}(z^*))^{-1}(\theta_a^* - \hat{\theta}_a(z^*)) \rangle| \preceq \frac{k(d + \log n)}{n}.$$

By (22) we have

$$| -\langle \theta_a^* - \theta_b^*, (\hat{\Sigma}(z^*))^{-1}(\theta_a^* - \theta_b^*) \rangle + \langle \theta_a^* - \theta_b^*, (\Sigma^*)^{-1}(\theta_a^* - \theta_b^*) \rangle | \preceq \sqrt{\frac{k(d + \log n)}{n}} \|\theta_a^* - \theta_b^*\|^2.$$

Using the results above we can get (19).

Finally we are going to establish (21). Recall the definition of $G_j(a, b, z)$ which has four terms. For the third and fourth terms, we have

$$| -\langle \theta_a^* - \hat{\theta}_b(z), (\hat{\Sigma}(z))^{-1}(\theta_a^* - \hat{\theta}_b(z)) \rangle + \langle \theta_a^* - \hat{\theta}_b(z^*), (\hat{\Sigma}(z))^{-1}(\theta_a^* - \hat{\theta}_b(z^*)) \rangle |$$

$$\preceq \|\hat{\theta}_b(z) - \hat{\theta}_b(z^*)\|^2 + \|\hat{\theta}_b(z) - \hat{\theta}_b(z^*)\| \|\theta_a^* - \theta_b^*\|,$$

and

$$| -\langle \theta_a^* - \hat{\theta}_b(z^*), (\hat{\Sigma}(z))^{-1}(\theta_a^* - \hat{\theta}_b(z^*)) \rangle + \langle \theta_a^* - \hat{\theta}_b(z^*), (\hat{\Sigma}(z^*))^{-1}(\theta_a^* - \hat{\theta}_b(z^*)) \rangle |$$

$$\preceq \|\theta_a^* - \theta_b^*\|^2 \|(\hat{\Sigma}(z))^{-1} - (\hat{\Sigma}(z^*))^{-1}\|.$$

We can easily verify that the other two terms are smaller than the above two terms. Then, by using (42) and (23), we have

$$\frac{|G_j(z_j^*, b, z)|}{\langle \theta_{z_j^*}^* - \theta_b^*, (\Sigma^*)^{-1}(\theta_{z_j^*}^* - \theta_b^*) \rangle}$$

$$\preceq \frac{\|\hat{\theta}_b(z) - \hat{\theta}_b(z^*)\|^2 + \|\hat{\theta}_b(z) - \hat{\theta}_b(z^*)\| \|\theta_{z_j^*}^* - \theta_b^*\| + \|\theta_{z_j^*}^* - \theta_b^*\|^2 \|(\hat{\Sigma}(z))^{-1} - (\hat{\Sigma}(z^*))^{-1}\|}{\|\theta_{z_j^*}^* - \theta_b^*\|^2}$$

$$\preceq \frac{k\tau}{n} + \frac{k}{\Delta} \sqrt{\frac{\tau}{n}} + \frac{kd\sqrt{\tau}}{n\Delta}.$$

$\square$

**Lemma A.5.** *With the same conditions as in Theorem 2.2, for any $\delta = o(1)$, we have*

$$\xi_{ideal}(\delta) \le n \exp\left(-(1 + o(1))\frac{SNR^2}{8}\right).$$

*with probability at least $1 - n^{-C'} - \exp(-SNR)$.*

*Proof.* Under the conditions of Theorem 2.2, the inequalities (33)-(38) hold with probability at least $1 - n^{-C'}$. In the remaining proof, we will work on the event these inequalities hold. Recall the definition of $\xi_{\text{ideal}}$. We can write

$$\xi_{\text{ideal}}(\delta) = \sum_{j=1}^{n} \sum_{b \in [k] \setminus \{z_j^*\}} \|\theta_{z_j^*}^* - \theta_b^*\|^2 \mathbb{I}\left\{ \langle \epsilon_j, (\hat{\Sigma}(z^*))^{-1}(\hat{\theta}_{z_j^*}(z^*) - \hat{\theta}_b(z^*)) \rangle \le -\frac{1-\delta}{2}\langle \theta_{z_j^*}^* - \theta_b^*, (\Sigma^*)^{-1}(\theta_{z_j^*}^* - \theta_b^*) \rangle \right\}$$

$$\le \sum_{j=1}^{n} \sum_{b \in [k] \setminus \{z_j^*\}} \|\theta_{z_j^*}^* - \theta_b^*\|^2 \mathbb{I}\left\{ \langle \epsilon_j, (\Sigma^*)^{-1}(\theta_{z_j^*}^* - \theta_b^*) \rangle \le -\frac{1-\delta-\bar\delta}{2}\langle \theta_{z_j^*}^* - \theta_b^*, (\Sigma^*)^{-1}(\theta_{z_j^*}^* - \theta_b^*) \rangle \right\}$$

$$+ \sum_{j=1}^{n} \sum_{b \in [k] \setminus \{z_j^*\}} \|\theta_{z_j^*}^* - \theta_b^*\|^2 \mathbb{I}\left\{ \langle \epsilon_j, ((\hat{\Sigma}(z^*))^{-1} - (\Sigma^*)^{-1})(\theta_{z_j^*}^* - \theta_b^*) \rangle \le -\frac{\bar\delta}{6}\langle \theta_{z_j^*}^* - \theta_b^*, (\Sigma^*)^{-1}(\theta_{z_j^*}^* - \theta_b^*) \rangle \right\}$$

$$+ \sum_{j=1}^{n} \sum_{b \in [k] \setminus \{z_j^*\}} \|\theta_{z_j^*}^* - \theta_b^*\|^2 \mathbb{I}\left\{ \langle \epsilon_j, (\hat{\Sigma}(z^*))^{-1}(\hat{\theta}_{z_j^*}(z^*) - \theta_{z_j^*}^*) \rangle \le -\frac{\bar\delta}{6}\langle \theta_{z_j^*}^* - \theta_b^*, (\Sigma^*)^{-1}(\theta_{z_j^*}^* - \theta_b^*) \rangle \right\}$$

$$+ \sum_{j=1}^{n} \sum_{b \in [k] \setminus \{z_j^*\}} \|\theta_{z_j^*}^* - \theta_b^*\|^2 \mathbb{I}\left\{ -\langle \epsilon_j, (\hat{\Sigma}(z^*))^{-1}(\hat{\theta}_b(z^*) - \theta_b^*) \rangle \le -\frac{\bar\delta}{6}\langle \theta_{z_j^*}^* - \theta_b^*, (\Sigma^*)^{-1}(\theta_{z_j^*}^* - \theta_b^*) \rangle \right\}$$

$$= M_1 + M_2 + M_3 + M_4.$$

where $\bar\delta = \bar\delta_n$ is some sequence to be chosen later. We bound the four terms sequentially. Suppose $\epsilon_j = (\Sigma^*)^{1/2} w_j$, where $w_j \overset{ind}{\sim} \mathcal{N}(0, I_d)$. By (22), we know

$$M_2 \le \sum_{j=1}^{n} \sum_{b \in [k] \setminus \{z_j^*\}} \|\theta_{z_j^*}^* - \theta_b^*\|^2 \mathbb{I}\left\{ \frac{\bar\delta}{6\lambda_{\max}} \|\theta_{z_j^*}^* - \theta_b^*\|^2 \le \lambda_{\max} \|w_j\| \|(\hat{\Sigma}(z^*))^{-1} - (\Sigma^*)^{-1}\| \|\theta_{z_j^*}^* - \theta_b^*\| \right\}$$

$$\le \sum_{j=1}^{n} \sum_{b \in [k] \setminus \{z_j^*\}} \|\theta_{z_j^*}^* - \theta_b^*\|^2 \mathbb{I}\left\{ C\bar\delta \|\theta_{z_j^*}^* - \theta_b^*\| \sqrt{\frac{n}{d + \log n}} \le \|w_j\| \right\}$$

$$\le \sum_{j=1}^{n} \sum_{b \in [k] \setminus \{z_j^*\}} \|\theta_{z_j^*}^* - \theta_b^*\|^2 \mathbb{I}\left\{ C\bar\delta^2 \|\theta_{z_j^*}^* - \theta_b^*\|^2 \frac{n}{d + \log n} - 2d \le \|w_j\|^2 - 2d \right\},$$

where $C$ is a constant which may vary from line by line. Recall that $kd = O(\sqrt{n})$, $\min_{a \ne b} \|\theta_a^* - \theta_b^*\| \to \infty$, and $\Delta/k \to \infty$ by assumption. Let $n^{-\frac{1}{4}} = o(\bar\delta)$. Using the $\chi^2$ tail probability in Lemma C.1, we have for any $a \ne b \in [k]$,

$$\mathbb{E}M_2 \le \sum_{j=1}^{n} \sum_{b \in [k] \setminus \{z_j^*\}} \|\theta_{z_j^*}^* - \theta_b^*\|^2 \exp\left(-C\bar\delta^2 \|\theta_{z_j^*}^* - \theta_b^*\|^2 \sqrt{n}\right) \le n \exp\left(-(1 + o(1))\frac{SNR^2}{8}\right).$$

We can obtain similar bounds on $M_3$ and $M_4$ by using (41). For $M_1$, the Gaussian tail bound leads to the inequality

$$\mathbb{P}\left\{ \langle \epsilon_j, (\Sigma^*)^{-1}(\theta_a^* - \theta_b^*) \rangle \le -\frac{1-\delta-\bar\delta}{2}\langle \theta_a^* - \theta_b^*, (\Sigma^*)^{-1}(\theta_a^* - \theta_b^*) \rangle \right\}$$

$$= \mathbb{P}\left\{ \langle w_j, (\Sigma^*)^{-1/2}(\theta_a^* - \theta_b^*) \rangle \le -\frac{1-\delta-\bar\delta}{2}\langle \theta_a^* - \theta_b^*, (\Sigma^*)^{-1}(\theta_a^* - \theta_b^*) \rangle \right\}$$

$$\le \exp\left(-\frac{(1-\delta-\bar\delta)^2}{8}\langle \theta_a^* - \theta_b^*, (\Sigma^*)^{-1}(\theta_a^* - \theta_b^*) \rangle\right).$$

Thus,

$$\mathbb{E} M_1 \leq \sum_{j=1}^{n} \sum_{b \in [k] \setminus \{z_j^*\}} \|\theta_{z_j^*}^* - \theta_b^*\|^2 \exp\left(-\frac{(1 - \delta - \bar{\delta})^2}{8} \langle \theta_a^* - \theta_b^*, (\Sigma^*)^{-1} (\theta_a^* - \theta_b^*) \rangle\right)$$

$$\leq n \exp\left(-(1 + o(1))\frac{\mathrm{SNR}^2}{8}\right).$$

Overall, we have $\mathbb{E} \xi_{\mathrm{ideal}} \preceq n \exp\left(-(1 + o(1))\frac{\mathrm{SNR}^2}{8}\right)$. By the Markov's inequality, we have

$$\mathbb{P}(\xi_{\mathrm{ideal}}(\delta_n) \geq \mathbb{E} \xi_{\mathrm{ideal}} \exp(\mathrm{SNR})) \leq \exp(-\mathrm{SNR}).$$

In other words, with probability at least $1 - \exp(-\mathrm{SNR})$, we have

$$\xi_{\mathrm{ideal}}(\delta_n) \leq \mathbb{E} \xi_{\mathrm{ideal}}(\delta_n) \exp(\mathrm{SNR}) \leq n \exp\left(-(1 + o(1))\frac{\mathrm{SNR}^2}{8}\right).$$

$\square$

*Proof of Theorem 2.2.* By Lemmas A.3 - A.5, we have that Conditions 1 - 5 are satisfied with probability at least $1 - \eta - n^{-1} - \exp(-\mathrm{SNR})$. Then applying Lemma A.3, we have

$$\ell(z^{(t)}, z^*) \leq n \exp\left(-(1 + o(1))\frac{\mathrm{SNR}^2}{8}\right) + \frac{1}{2} \ell(z^{(t-1)}, z^*), \quad \text{for all } t \geq 1.$$

By (16), and since there exists a constant $C$ such that $\Delta \leq C\mathrm{SNR}$, we can conclude

$$h(z^{(t)}, z^*) \leq \exp\left(-(1 + o(1))\frac{\mathrm{SNR}^2}{8}\right) + 2^{-t}, \quad \text{for all } t \geq 1.$$

Notice that $h(\cdot, \cdot)$ takes value in the set $\{j/n : j \in [n] \cup \{0\}\}$, the term $2^{-t}$ in the above inequality should be negligible as long as $2^{-t} = o(n^{-1})$. Thus, we can claim

$$h(z^{(t)}, z^*) \leq \exp\left(-(1 + o(1))\frac{\mathrm{SNR}^2}{8}\right), \quad \text{for all } t \geq \log n.$$

$\square$

# B  Proofs in Section 3

## B.1  Proofs for the Lower Bound

*Proof of Lemma 3.1.* The Neyman-Pearson lemma tells us the likelihood ratio test $\phi$ is the optimal procedure. Following the proof of Lemma A.1, we have

$$\mathbb{P}_{\mathbb{H}_0}(\phi = 1) + \mathbb{P}_{\mathbb{H}_1}(\phi = 0) = \mathbb{P}(\epsilon \in B_{1,2}) + \mathbb{P}(\epsilon \in B_{2,1})$$

$$\geq \exp\left(-\frac{1 + o(1)}{8} \mathrm{SNR}_{1,2}'^2\right) + \exp\left(-\frac{1 + o(1)}{8} \mathrm{SNR}_{2,1}'^2\right),$$

where the last inequality is by Lemma C.10. $\square$

*Proof of Theorem 3.1.* The proof is identical to the proof of Theorem 2.1 and is omitted here. $\square$

## B.2  Proofs for the Upper Bound

We adopt a similar proof idea as in Section 2 for Model 1. We first present an error decomposition for the one-step analysis for Algorithm 2. In Lemma B.1, we show the loss decays after a one-step iteration under Conditions 6 - 11. Then in Lemma B.2 we extend the result to multiple iterations, under two extra Conditions 12 - 13. Finally, we show that all the conditions are satisfied with high probability and thus prove Theorem 3.2.

In the statement of Theorem 3.2, we assume $\max_{a,b\in[k]} \lambda_d(\Sigma_a^*)/\lambda_1(\Sigma_b^*) = O(1)$ for the covariance matrices $\{\Sigma_a^*\}_{a\in[k]}$. Without loss of generality, we can replace it by assuming $\{\Sigma_a^*\}_{a\in[k]}$ satisfy

$$\lambda_{\min} \leq \min_{a\in[k]} \lambda_1(\Sigma_a^*) \leq \max_{a\in[k]} \lambda_d(\Sigma_a^*) \leq \lambda_{\max} \tag{24}$$

where $\lambda_{\min}, \lambda_{\max} > 0$ are two constants. This is due to the scaling properties of the normal distributions. The reasoning is the same as that in (15) for Model 1 and is omitted here. For the remainder of this section, we will assume that (24) holds for the covariance matrices.

**Error Decomposition for the One-step Analysis:** Consider an arbitrary $z \in [k]^n$. Apply (12), (13), and (14) on $z$ to obtain $\{\hat{\theta}_a(z)\}_{a\in[k]}$, $\{\hat{\Sigma}_a(z)\}_{a\in[k]}$, and $\hat{z}(z)$:

$$\hat{\theta}_a(z) = \frac{\sum_{j\in[n]} Y_j \mathbb{I}\{z_j = a\}}{\sum_{j\in[n]} \mathbb{I}\{z_j = a\}},$$

$$\hat{\Sigma}_a(z) = \frac{\sum_{j\in[n]}(Y_j - \hat{\theta}_a(z))(Y_j - \hat{\theta}_a(z))^T \mathbb{I}\{z_j = a\}}{\sum_{j\in[n]} \mathbb{I}\{z_j = a\}}, \quad \forall a \in [k],$$

$$\hat{z}_j(t) = \underset{a\in[k]}{\operatorname{argmin}} (Y_j - \hat{\theta}_a(z))^T (\hat{\Sigma}_a(a))^{-1}(Y_j - \hat{\theta}_a(z)) + \log|\hat{\Sigma}_a(z)|, \quad \forall j \in [n].$$

For simplicity, we denote $\hat{z}$ as shorthand for $\hat{z}(z)$. Let $j \in [n]$ be an arbitrary index with $z_j^* = a$. According to (14), $z_j^*$ will be incorrectly estimated after one iteration in $\hat{z}$ if $a \neq \operatorname{argmin}_{b\in[k]}(Y_j - \hat{\theta}_b(z))^T(\hat{\Sigma}_b(z))^{-1}(Y_j - \hat{\theta}_b(z)) + \log|\hat{\Sigma}_b(z)|$. That is, it is important to analyze the event

$$\langle Y_j - \hat{\theta}_b(z), (\hat{\Sigma}_b(z))^{-1}(Y_j - \hat{\theta}_b(z))\rangle + \log|\hat{\Sigma}_b(z)| \leq \langle Y_j - \hat{\theta}_a(z), (\hat{\Sigma}_a(z))^{-1}(Y_j - \hat{\theta}_a(z))\rangle + \log|\hat{\Sigma}_a(z)|, \tag{25}$$

for any $b \in [k] \setminus \{a\}$. After some rearrangements, we can see (25) is equivalent to

$$\langle \epsilon_j, (\hat{\Sigma}_b(z^*))^{-1}(\theta_a^* - \hat{\theta}_b(z^*))\rangle - \langle \epsilon_j, (\hat{\Sigma}_a(z^*))^{-1}(\theta_a^* - \hat{\theta}_a(z^*))\rangle$$

$$+ \frac{1}{2}\langle \epsilon_j, ((\hat{\Sigma}_b(z^*))^{-1} - (\hat{\Sigma}_a(z^*))^{-1})\epsilon_j\rangle - \frac{1}{2}\log|\Sigma_a^*| + \frac{1}{2}\log|\Sigma_b^*|$$

$$\leq -\frac{1}{2}\langle \theta_a^* - \theta_b^*, (\Sigma_b^*)^{-1}(\theta_a^* - \theta_b^*)\rangle$$

$$+ F_j(a, b, z) + Q_j(a, b, z) + G_j(a, b, z) + H_j(a, b, z) + K_j(a, b, z) + L_j(a, b, z),$$

where

$$F_j(a, b, z) = \langle \epsilon_j, (\hat{\Sigma}_b(z))^{-1}(\hat{\theta}_b(z) - \hat{\theta}_b(z^*))\rangle - \langle \epsilon_j, (\hat{\Sigma}_a(z))^{-1}(\hat{\theta}_a(z) - \hat{\theta}_a(z^*))\rangle$$

$$- \langle \epsilon_j, ((\hat{\Sigma}_b(z))^{-1} - (\hat{\Sigma}_b(z^*))^{-1})(\theta_a^* - \hat{\theta}_b(z^*))\rangle$$

$$+ \langle \epsilon_j, ((\hat{\Sigma}_a(z))^{-1} - (\hat{\Sigma}_a(z^*))^{-1})(\theta_a^* - \hat{\theta}_a(z^*))\rangle,$$

$$Q_j(a, b, z) = -\frac{1}{2}\langle \epsilon_j, ((\hat{\Sigma}_b(z))^{-1} - (\hat{\Sigma}_b(z^*))^{-1})\epsilon_j\rangle + \frac{1}{2}\langle \epsilon_j, ((\hat{\Sigma}_a(z))^{-1} - (\hat{\Sigma}_a(z^*))^{-1})\epsilon_j\rangle,$$

$$G_j(a, b, z) = \frac{1}{2}\langle \theta_a^* - \hat{\theta}_a(z), (\hat{\Sigma}_a(z))^{-1}(\theta_a^* - \hat{\theta}_a(z))\rangle - \frac{1}{2}\langle \theta_a^* - \hat{\theta}_a(z^*), (\hat{\Sigma}_a(z))^{-1}(\theta_a^* - \hat{\theta}_a(z^*))\rangle$$

$$+ \frac{1}{2}\langle \theta_a^* - \hat{\theta}_a(z^*), (\hat{\Sigma}_a(z))^{-1}(\theta_a^* - \hat{\theta}_a(z^*))\rangle - \frac{1}{2}\langle \theta_a^* - \hat{\theta}_a(z^*), (\hat{\Sigma}_a(z^*))^{-1}(\theta_a^* - \hat{\theta}_a(z^*))\rangle$$

$$- \frac{1}{2}\langle \theta_a^* - \hat{\theta}_b(z), (\hat{\Sigma}_b(z))^{-1}(\theta_a^* - \hat{\theta}_b(z))\rangle + \frac{1}{2}\langle \theta_a^* - \hat{\theta}_b(z^*), (\hat{\Sigma}_b(z))^{-1}(\theta_a^* - \hat{\theta}_b(z^*))\rangle$$

$$- \frac{1}{2}\langle \theta_a^* - \hat{\theta}_b(z^*), (\hat{\Sigma}_b(z))^{-1}(\theta_a^* - \hat{\theta}_b(z^*))\rangle + \frac{1}{2}\langle \theta_a^* - \hat{\theta}_b(z^*), (\hat{\Sigma}_b(z^*))^{-1}(\theta_a^* - \hat{\theta}_b(z^*))\rangle,$$

$$H_j(a, b, z) = -\frac{1}{2}\langle \theta_a^* - \hat{\theta}_b(z^*), (\hat{\Sigma}_b(z^*))^{-1}(\theta_a^* - \hat{\theta}_b(z^*))\rangle + \frac{1}{2}\langle \theta_a^* - \theta_b^*, (\hat{\Sigma}_b(z^*))^{-1}(\theta_a^* - \theta_b^*)\rangle$$

$$- \frac{1}{2}\langle \theta_a^* - \theta_b^*, (\hat{\Sigma}_b(z^*))^{-1}(\theta_a^* - \theta_b^*)\rangle + \frac{1}{2}\langle \theta_a^* - \theta_b^*, (\Sigma_b^*)^{-1}(\theta_a^* - \theta_b^*)\rangle$$

$$+ \frac{1}{2}\langle \theta_a^* - \hat{\theta}_a(z^*), (\hat{\Sigma}_a(z^*))^{-1}(\theta_a^* - \hat{\theta}_a(z^*))\rangle,$$

$$K_j(a, b, z) = \frac{1}{2}(\log |\hat{\Sigma}_a(z)| - \log |\hat{\Sigma}_a(z^*)|) - \frac{1}{2}(\log |\hat{\Sigma}_b(z)| - \log |\hat{\Sigma}_b(z^*)|),$$

$$L_j(a, b, z) = \frac{1}{2}(\log |\hat{\Sigma}_a(z^*)| - \log |\Sigma_a^*|) - \frac{1}{2}(\log |\hat{\Sigma}_b(z^*)| - \log |\Sigma_b^*|).$$

Among these terms, $F_j, G_j, H_j$ are nearly identical to their counterparts in Section A.2.2 with $\hat{\Sigma}(z)$ replaced by $\hat{\Sigma}_a(z)$ or $\hat{\Sigma}_b(z)$. There are three extra terms not appearing in Section A.2.2: $Q_j$ is a quadratic term of $\epsilon_j$ and $K_j, L_j$ are terms involving matrix determinants.

**Conditions and Guarantees for One-step Analysis.** To establish the guarantee for the one-step analysis, we first give several conditions on the error terms.

**Condition 6.** *Assume that*

$$\max_{\{z:\ell(z,z^*)\leq\tau\}} \max_{j\in[n]} \max_{b\in[k]\setminus\{z_j^*\}} \frac{|H_j(z_j^*, b, z)|}{\langle \theta_{z_j}^* - \theta_b^*, (\Sigma_b^*)^{-1}(\theta_{z_j}^* - \theta_b^*)\rangle} \leq \frac{\delta}{12}$$

*holds with probability at least $1 - \eta_1$ for some $\tau, \delta, \eta_1 > 0$.*

**Condition 7.** *Assume that*

$$\max_{\{z:\ell(z,z^*)\leq\tau\}} \sum_{j=1}^{n} \max_{b\in[k]\setminus\{z_j^*\}} \frac{F_j(z_j^*, b, z)^2 \|\theta_{z_j^*}^* - \theta_b^*\|^2}{\langle \theta_{z_j^*}^* - \theta_b^*, (\Sigma_b^*)^{-1}(\theta_{z_j^*}^* - \theta_b^*)\rangle^2 \ell(z, z^*)} \leq \frac{\delta^2}{288}$$

*holds with probability at least $1 - \eta_2$ for some $\tau, \delta, \eta_2 > 0$.*

**Condition 8.** *Assume that*

$$\max_{\{z:\ell(z,z^*)\leq\tau\}} \max_{j\in[n]} \max_{b\in[k]\setminus\{z_j^*\}} \frac{|G_j(z_j^*, b, z)|}{\langle \theta_{z_j}^* - \theta_b^*, (\Sigma_b^*)^{-1}(\theta_{z_j}^* - \theta_b^*)\rangle} \leq \frac{\delta}{12}$$

*holds with probability at least $1 - \eta_3$ for some $\tau, \delta, \eta_3 > 0$.*

**Condition 9.** *Assume that*

$$\max_{\{z:\ell(z,z^*)\leq\tau\}} \sum_{j=1}^{n} \max_{b\in[k]\setminus\{z_j^*\}} \frac{Q_j(z_j^*, b, z)^2 \|\theta_{z_j^*}^* - \theta_b^*\|^2}{\langle \theta_{z_j^*}^* - \theta_b^*, (\Sigma_b^*)^{-1}(\theta_{z_j^*}^* - \theta_b^*)\rangle^2 \ell(z, z^*)} \leq \frac{\delta^2}{288}$$

*holds with probability at least $1 - \eta_4$ for some $\tau, \delta, \eta_4 > 0$.*

**Condition 10.** *Assume that*

$$\max_{\{z:\ell(z,z^*)\leq\tau\}} \sum_{j=1}^{n} \max_{b\in[k]\setminus\{z_j^*\}} \frac{K_j(z_j^*, b, z)^2 \|\theta_{z_j^*}^* - \theta_b^*\|^2}{\langle \theta_{z_j^*}^* - \theta_b^*, (\Sigma_b^*)^{-1}(\theta_{z_j^*}^* - \theta_b^*)\rangle^2 \ell(z, z^*)} \leq \frac{\delta^2}{288}$$

*holds with probability at least $1 - \eta_5$ for some $\tau, \delta, \eta_5 > 0$.*

**Condition 11.** *Assume that*

$$\max_{\{z:\ell(z,z^*)\leq\tau\}} \max_{j\in[n]} \max_{b\in[k]\setminus\{z_j^*\}} \frac{|L_j(z_j^*, b, z)|}{\langle \theta_{z_j}^* - \theta_b^*, (\Sigma_b^*)^{-1}(\theta_{z_j}^* - \theta_b^*)\rangle} \leq \frac{\delta}{12}$$

*holds with probability at least $1 - \eta_6$ for some $\tau, \delta, \eta_6 > 0$.*

We next define a quantity referred to as the ideal error,

$$\xi_{\text{ideal}}(\delta) = \sum_{j=1}^{n} \sum_{b\in[k]\setminus\{z_j^*\}} \|\theta_{z_j^*}^* - \theta_b^*\|^2 \mathbb{I}\{\langle \epsilon_j, (\hat{\Sigma}_b(z^*))^{-1}(\theta_a^* - \hat{\theta}_b(z^*))\rangle - \langle \epsilon_j, (\hat{\Sigma}_a(z^*))^{-1}(\theta_a^* - \hat{\theta}_a(z^*))\rangle$$

$$+ \frac{1}{2}\langle \epsilon_j, ((\hat{\Sigma}_b(z^*))^{-1} - (\hat{\Sigma}_a(z^*))^{-1})\epsilon_j\rangle - \frac{1}{2}\log|\Sigma_a^*| + \frac{1}{2}\log|\Sigma_b^*| \leq -\frac{1-\delta}{2}\langle \theta_a^* - \theta_b^*, (\Sigma_b^*)^{-1}(\theta_a^* - \theta_b^*)\rangle\}.$$

**Lemma B.1.** *Assumes Conditions 6 - 11 hold for some $\tau, \delta, \eta_1, \ldots, \eta_6 > 0$. We then have*

$$\mathbb{P}\left(\ell(\hat{z}, z^*) \leq \xi_{\text{ideal}}(\delta) + \frac{1}{2}\ell(z, z^*) \text{ for any } z \in [k]^n \text{ such that } \ell(z, z^*) \leq \tau\right) \geq 1 - \eta,$$

*where $\eta = \sum_{i=1}^{6} \eta_i$.*

*Proof.* The proof of this lemma is quite similar to the proof of Lemma A.2. The additional terms $Q_j$ and $K_j$ can be handled in the same way as $F_j$ while $L_j$ can be handled similarly to $H_j$. We omit the details here. $\square$

**Conditions and Guarantees for Multiple Iterations.** In the above, we establish a statistical guarantee for the one-step analysis. Now we will extend the result to multiple iterations. That is, starting from some initialization $z^{(0)}$, we will characterize how the losses $\ell(z^{(0)}, z^*)$, $\ell(z^{(1)}, z^*)$, $\ell(z^{(2)}, z^*)$, ..., decay. We impose conditions on $\xi_{\text{ideal}}(\delta)$ and the initialization $z^{(0)}$.

**Condition 12.** *Assume that*

$$\xi_{ideal}(\delta) \leq \frac{\tau}{2}$$

*holds with probability at least $1 - \eta_7$ for some $\tau, \delta, \eta_7 > 0$.*

Finally, we need a condition on the initialization.

**Condition 13.** *Assume that*

$$\ell(z^{(0)}, z^*) \leq \tau$$

*holds with probability at least $1 - \eta_8$ for some $\tau, \eta_8 > 0$.*

With these conditions satisfied, we can give a lemma that shows the convergence of our algorithm.

**Lemma B.2.** *Assumes Conditions 6 - 13 hold for some $\tau, \delta, \eta_1, \ldots, \eta_8 > 0$. We then have*

$$\ell(z^{(t)}, z^*) \leq \xi_{ideal}(\delta) + \frac{1}{2}\ell(z^{(t-1)}, z^*)$$

*for all $t \geq 1$, with probability at least $1 - \eta$, where $\eta = \sum_{i=1}^{8} \eta_i$.*

*Proof.* The proof of this lemma is the same as the proof of Lemma A.3. $\qquad\square$

**With-high-probability Results for the Conditions and Proof of Theorem 3.2.** Lemma B.3 and Lemma B.4 are the counterparts of Lemmas A.4 and A.5 in Appendix A.2.2. Recall that (24) is assumed. By Lemma C.10, we have $\Delta$ is of the same order as $\text{SNR}'$, which will play a similar role as (18) in Section A.2.2.

Lemma B.3 and Lemma B.4 are counterparts of Lemmas A.4 and A.5 in Section A.2.2. The first lemma is for Conditions 6-11, providing upper bounds for the quantities involved in these conditions, showing that $\delta$ can be taken as some $o(1)$ term. The second lemma shows that for any $\delta = o(1)$, $\xi_{\text{ideal}}(\delta)$ is upper bounded by the desired minimax rate multiplied by the sample size $n$.

**Lemma B.3.** *Under the same conditions as in Theorem 3.2, for any constant $C' > 0$, there exists some constant $C > 0$ only depending on $\alpha, C', \lambda_{\min}, \lambda_{\max}$ such that*

$$\max_{\{z:\ell(z,z^*)\leq\tau\}} \max_{j\in[n]} \max_{b\in[k]\backslash\{z_j^*\}} \frac{|H_j(z_j^*, b, z)|}{\langle\theta_{z_j^*}^* - \theta_b^*, (\Sigma_b^*)^{-1}(\theta_{z_j^*}^* - \theta_b^*)\rangle} \leq C\sqrt{\frac{k(d + \log n)}{n}} \qquad (26)$$

$$\max_{\{z:\ell(z,z^*)\leq\tau\}} \sum_{j=1}^{n} \max_{b\in[k]\backslash\{z_j^*\}} \frac{F_j(z_j^*, b, z)^2\|\theta_{z_j^*}^* - \theta_b^*\|^2}{\langle\theta_{z_j^*}^* - \theta_b^*, (\Sigma_b^*)^{-1}(\theta_{z_j^*}^* - \theta_b^*)\rangle^2\ell(z,z^*)} \leq Ck^3\left(\frac{\tau}{n} + \frac{1}{\Delta^2} + \frac{d^2}{n\Delta^2}\right)$$

$$(27)$$

$$\max_{\{z:\ell(z,z^*)\leq\tau\}} \max_{j\in[n]} \max_{b\in[k]\backslash\{z_j^*\}} \frac{|G_j(z_j^*, b, z)|}{\langle\theta_{z_j^*}^* - \theta_b^*, (\Sigma_b^*)^{-1}(\theta_{z_j^*}^* - \theta_b^*)\rangle} \leq Ck\left(\frac{\tau}{n} + \frac{1}{\Delta}\sqrt{\frac{\tau}{n}} + \frac{d\sqrt{\tau}}{n\Delta}\right)$$

$$(28)$$

$$\max_{\{z:\ell(z,z^*)\leq\tau\}} \sum_{j=1}^{n} \max_{b\in[k]\backslash\{z_j^*\}} \frac{Q_j(z_j^*, b, z)^2\|\theta_{z_j^*}^* - \theta_b^*\|^2}{\langle\theta_{z_j^*}^* - \theta_b^*, (\Sigma_b^*)^{-1}(\theta_{z_j^*}^* - \theta_b^*)\rangle^2\ell(z,z^*)} \leq C\frac{k^3d^2}{\Delta^2}\left(\frac{\tau}{n} + \frac{1}{\Delta^2} + \frac{d^2}{n\Delta^2}\right)$$

$$(29)$$

$$\max_{\{z:\ell(z,z^*)\leq\tau\}} \sum_{j=1}^{n} \max_{b\in[k]\backslash\{z_j^*\}} \frac{K_j(z_j^*, b, z)^2\|\theta_{z_j^*}^* - \theta_b^*\|^2}{\langle\theta_{z_j^*}^* - \theta_b^*, (\Sigma_b^*)^{-1}(\theta_{z_j^*}^* - \theta_b^*)\rangle^2\ell(z,z^*)} \leq C\frac{k^3d^2}{\Delta^2}\left(\frac{\tau}{n} + \frac{1}{\Delta^2} + \frac{d^2}{n\Delta^2}\right)$$

$$(30)$$

$$\max_{\{z:\ell(z,z^*)\leq\tau\}} \max_{j\in[n]} \max_{b\in[k]\backslash\{z_j^*\}} \frac{|L_j(z_j^*, b, z)|}{\langle\theta_{z_j^*}^* - \theta_b^*, (\Sigma_b^*)^{-1}(\theta_{z_j^*}^* - \theta_b^*)\rangle} \leq C\frac{d}{\Delta^2}\sqrt{\frac{k(d + \log n)}{n}} \quad (31)$$

*with probability at least $1 - n^{-C'} - \frac{4}{nd}$. As a result, Conditions 6-11 hold for some $\delta = o(1)$.*

*Proof.* Under the conditions of Theorem 3.2, the inequalities (33)-(38) hold with probability at least $1 - n^{-C'}$. In the remaining proof, we will work on the event these inequalities hold. Hence, we can use the results from Lemma C.7 and C.8. Using the same arguments as in the proof of Lemma A.4, we can get (26), (27) and (28).

As for (29), we first use Lemma C.2 to have $\sum_{j=1}^{n} \|\epsilon_j\|^4 \leq 3nd$ with probability at least $1 - 4/(nd)$. Then, we have

$$
\sum_{j=1}^{n} \max_{b \in [k] \setminus \{z_j^*\}} \frac{Q_j(z_j^*, b, z)^2 \|\theta_{z_j^*}^* - \theta_b^*\|^2}{\langle \theta_{z_j^*}^* - \theta_b^*, (\Sigma_b^*)^{-1}(\theta_{z_j^*}^* - \theta_b^*) \rangle^2 \ell(z, z^*)} \preceq \sum_{j=1}^{n} \sum_{b=1}^{k} \frac{Q_j(z_j^*, b, z)^2}{\Delta^2 \ell(z, z^*)}
$$

$$
\leq k \sum_{j=1}^{n} \|\epsilon_j\|^4 \frac{\max_{a \in [k]} \|(\hat{\Sigma}_a(z))^{-1} - (\hat{\Sigma}_a(z^*))^{-1}\|^2}{\Delta^2 \ell(z, z^*)}
$$

$$
\preceq \frac{k^3 d^2}{\Delta^2} \left( \frac{\tau}{n} + \frac{1}{\Delta^2} + \frac{d^2}{n\Delta^2} \right),
$$

where the last inequality is due to (53) and the fact that $\ell(z, z^*) \leq \tau$.

Next for (30), notice that by (43), (44), and SNR$' \to \infty$, we have for any $1 \leq i \leq d$, $\frac{\lambda_{\min}}{2} \leq \lambda_i(\hat{\Sigma}_a(z^*)) \leq 2\lambda_{\max}$ and

$$
\left| \log(1 + \max_{a \in [k]} \frac{\|\hat{\Sigma}_a(z) - \hat{\Sigma}_a(z^*)\|_2}{\lambda_i(\hat{\Sigma}_a(z^*))}) \right| \leq \left| \log(1 - \max_{a \in [k]} \frac{\|\hat{\Sigma}_a(z) - \hat{\Sigma}_a(z^*)\|_2}{\lambda_i(\hat{\Sigma}_a(z^*))}) \right|.
$$

Thus by Weyl's inequality, we know

$$
\max_{a \in [k]} \left| \log |\hat{\Sigma}_a(z)| - \log |\hat{\Sigma}_a(z^*)| \right|
$$

$$
= \max_{a \in [k]} \left| \log \frac{|\hat{\Sigma}_a(z)|}{|\hat{\Sigma}_a(z^*)|} \right|
$$

$$
\leq \left| \sum_{i=1}^{d} \log(1 - \frac{\max_{a \in [k]} \|\hat{\Sigma}_a(z) - \hat{\Sigma}_a(z^*)\|_2}{\lambda_i(\hat{\Sigma}_a(z^*))}) \right|
$$

$$
\leq \sum_{i=1}^{d} \log \left( 1 + \max_{a \in [k]} \frac{\|\hat{\Sigma}_a(z) - \hat{\Sigma}_a(z^*)\|_2}{\lambda_i(\hat{\Sigma}_a(z^*))} + \frac{\max_{a \in [k]} \frac{\|\hat{\Sigma}_a(z) - \hat{\Sigma}_a(z^*)\|_2^2}{\lambda_i^2(\hat{\Sigma}_a(z^*))}}{1 - \max_{a \in [k]} \frac{\|\hat{\Sigma}_a(z) - \hat{\Sigma}_a(z^*)\|_2}{\lambda_i(\hat{\Sigma}_a(z^*))}} \right)
$$

$$
\preceq d \|\hat{\Sigma}_a(z) - \hat{\Sigma}_a(z^*)\|_2, \tag{32}
$$

where the last inequality is due to the fact that $\lambda_i(\hat{\Sigma}_a(z^*))$ is at the constant rate, $\|\hat{\Sigma}_a(z) - \hat{\Sigma}_a(z^*)\|_2 = o(1)$ and the inequality $\log(1 + x) \leq x$ for any $x > 0$. (32) yields to the inequality

$$
\sum_{j=1}^{n} \max_{b \in [k] \setminus \{z_j^*\}} \frac{K_j(z_j^*, b, z)^2 \|\theta_{z_j^*}^* - \theta_b^*\|^2}{\langle \theta_{z_j^*}^* - \theta_b^*, (\Sigma_b^*)^{-1}(\theta_{z_j^*}^* - \theta_b^*) \rangle^2 \ell(z, z^*)} \preceq \sum_{j=1}^{n} \frac{d^2 \max_{a \in [k]} \|\hat{\Sigma}_a(z) - \hat{\Sigma}_a(z^*)\|^2}{\Delta^2 \ell(z, z^*)}
$$

$$
\preceq \frac{k^2 d^2}{\Delta^2} \left( \frac{\tau}{n} + \frac{1}{\Delta^2} + \frac{d^2}{n\Delta^2} \right).
$$

Finally for (31), by (43) and the similar argument as (32), we can get

$$
\max_{a \in [k]} \left| \log |\hat{\Sigma}_a(z^*)| - \log |\Sigma_a^*| \right| \preceq d \sqrt{\frac{k(d + \log n)}{n}}
$$

which implies (31). We complete the proof. $\qquad \square$

**Lemma B.4.** *With the same conditions as Theorem 3.2, for any sequence $\delta_n = o(1)$, we have*

$$
\xi_{ideal}(\delta_n) \leq n \exp\left( -(1 + o(1)) \frac{SNR'^2}{8} \right).
$$

*with probability at least $1 - n^{-C'} - \exp(-SNR')$.*

*Proof.* Under the conditions of Theorem 3.2, the inequalities (33)-(38) hold with probability at least $1 - n^{-C'}$. In the remaining proof, we will work on the event these inequalities hold. Similar to the proof of Lemma A.5, we have a decomposition $\xi_{\text{ideal}} \leq \sum_{i=1}^{6} M_i$ where

$$M_1 = \sum_{j=1}^{n} \sum_{b \in [k] \setminus \{z_j^*\}} \left\| \theta_{z_j^*}^* - \theta_b^* \right\|^2 \mathbb{I} \bigg\{ \langle \epsilon_j, (\Sigma_b^*)^{-1} (\theta_{z_j^*}^* - \theta_b^*) \rangle + \frac{1}{2} \langle \epsilon_j, ((\Sigma_b^*)^{-1} - (\Sigma_{z_j^*}^*)^{-1}) \epsilon_j \rangle$$
$$- \frac{1}{2} \log |\Sigma_{z_j^*}^*| + \frac{1}{2} \log |\Sigma_b^*| \leq -\frac{1 - \delta - \bar{\delta}}{2} \langle \theta_{z_j^*}^* - \theta_b^*, (\Sigma_b^*)^{-1} (\theta_{z_j^*}^* - \theta_b^*) \rangle \bigg\}$$

is the main term and

$$M_2 = \sum_{j=1}^{n} \sum_{b \in [k] \setminus \{z_j^*\}} \left\| \theta_{z_j^*}^* - \theta_b^* \right\|^2 \mathbb{I} \bigg\{ \langle \epsilon_j, ((\hat{\Sigma}_b(z^*))^{-1} - (\Sigma_b^*)^{-1}) (\theta_{z_j^*}^* - \theta_b^*) \rangle \leq -\frac{\bar{\delta}}{10} \langle \theta_{z_j^*}^* - \theta_b^*, (\Sigma_b^*)^{-1} (\theta_{z_j^*}^* - \theta_b^*) \rangle \bigg\}$$

$$M_3 = \sum_{j=1}^{n} \sum_{b \in [k] \setminus \{z_j^*\}} \left\| \theta_{z_j^*}^* - \theta_b^* \right\|^2 \mathbb{I} \bigg\{ -\langle \epsilon_j, (\hat{\Sigma}_{z_j^*}(z^*))^{-1} (\theta_{z_j^*}^* - \hat{\theta}_{z_j^*}(z^*)) \rangle \leq -\frac{\bar{\delta}}{10} \langle \theta_{z_j^*}^* - \theta_b^*, (\Sigma_b^*)^{-1} (\theta_{z_j^*}^* - \theta_b^*) \rangle \bigg\}$$

$$M_4 = \sum_{j=1}^{n} \sum_{b \in [k] \setminus \{z_j^*\}} \left\| \theta_{z_j^*}^* - \theta_b^* \right\|^2 \mathbb{I} \bigg\{ -\langle \epsilon_j, (\hat{\Sigma}_b(z^*))^{-1} (\hat{\theta}_b(z^*) - \theta_b^*) \rangle \leq -\frac{\bar{\delta}}{10} \langle \theta_{z_j^*}^* - \theta_b^*, (\Sigma_b^*)^{-1} (\theta_{z_j^*}^* - \theta_b^*) \rangle \bigg\}$$

$$M_5 = \sum_{j=1}^{n} \sum_{b \in [k] \setminus \{z_j^*\}} \left\| \theta_{z_j^*}^* - \theta_b^* \right\|^2 \mathbb{I} \bigg\{ \frac{1}{2} \langle \epsilon_j, ((\hat{\Sigma}_b(z^*))^{-1} - (\Sigma_b^*)^{-1}) \epsilon_j \rangle \leq -\frac{\bar{\delta}}{10} \langle \theta_{z_j^*}^* - \theta_b^*, (\Sigma_b^*)^{-1} (\theta_{z_j^*}^* - \theta_b^*) \rangle \bigg\}$$

$$M_6 = \sum_{j=1}^{n} \sum_{b \in [k] \setminus \{z_j^*\}} \left\| \theta_{z_j^*}^* - \theta_b^* \right\|^2 \mathbb{I} \bigg\{ -\frac{1}{2} \langle \epsilon_j, ((\hat{\Sigma}_{z_j^*}(z^*))^{-1} - (\Sigma_{z_j^*}^*)^{-1}) \epsilon_j \rangle \leq -\frac{\bar{\delta}}{10} \langle \theta_{z_j^*}^* - \theta_b^*, (\Sigma_b^*)^{-1} (\theta_{z_j^*}^* - \theta_b^*) \rangle \bigg\}.$$

Using the same arguments as the proof of Lemma A.5, we can choose some $\bar{\delta} = \bar{\delta}_n = o(1)$ which is slowly diverging to zero satisfying

$$\mathbb{E} M_i \leq n \exp \left( -(1 + o(1)) \frac{\text{SNR}'^2}{2} \right) \quad \text{for } i = 2, 3, 4.$$

As for $M_5$, by (43) we have

$$M_5 \leq \sum_{j=1}^{n} \sum_{b \in [k] \setminus \{z_j^*\}} \left\| \theta_{z_j^*}^* - \theta_b^* \right\|^2 \mathbb{I} \bigg\{ C \bar{\delta} \left\| \theta_{z_j^*}^* - \theta_b^* \right\|^2 \leq \|w_j\|^2 \sqrt{\frac{\log n}{n}} \bigg\},$$

where $C$ is a constant and $w_j \overset{iid}{\sim} \mathcal{N}(0, I_d)$. Since there exists some constant $C'$ such that $\text{SNR}' \leq C'\Delta$, we can choose appropriate $\bar{\delta} = o(1)$ such that

$$\mathbb{E} M_5 \leq \sum_{j=1}^{n} \sum_{b \in [k] \setminus \{z_j^*\}} \left\| \theta_{z_j^*}^* - \theta_b^* \right\|^2 \mathbb{P} \bigg\{ C \bar{\delta} \left\| \theta_{z_j^*}^* - \theta_b^* \right\|^2 \sqrt{\frac{n}{\log n}} \leq \|w_j\|^2 \bigg\}$$
$$\leq n \exp \left( -(1 + o(1)) \frac{\text{SNR}'^2}{8} \right).$$

$M_6$ is essentially the same with $M_5$ and can be proved similarly. Finally for $M_1$, using Lemma C.10, we have

$$
\mathbb{P}\bigg( \langle \epsilon_j, (\Sigma_b^*)^{-1}(\theta_{z_j^*}^* - \theta_b^*) \rangle + \frac{1}{2}\langle \epsilon_j, ((\Sigma_b^*)^{-1} - (\Sigma_{z_j^*}^*)^{-1})\epsilon_j \rangle
$$
$$
- \frac{1}{2}\log|\Sigma_{z_j^*}^*| + \frac{1}{2}\log|\Sigma_b^*| \le -\frac{1-\delta-\bar{\delta}}{2}\langle \theta_{z_j^*}^* - \theta_b^*, (\Sigma_b^*)^{-1}(\theta_{z_j^*}^* - \theta_b^*) \rangle \bigg)
$$
$$
=\mathbb{P}\bigg( \langle w_j, (\Sigma_{z_j^*}^*)^{\frac{1}{2}}(\Sigma_b^*)^{-1}(\theta_{z_j^*}^* - \theta_b^*) \rangle + \frac{1}{2}\langle w_j, ((\Sigma_{z_j^*}^*)^{\frac{1}{2}}(\Sigma_b^*)^{-1}(\Sigma_{z_j^*}^*)^{\frac{1}{2}} - I_d)w_j \rangle
$$
$$
- \frac{1}{2}\log|\Sigma_{z_j^*}^*| + \frac{1}{2}\log|\Sigma_b^*| \le -\frac{1-\delta-\bar{\delta}}{2}\langle \theta_{z_j^*}^* - \theta_b^*, (\Sigma_b^*)^{-1}(\theta_{z_j^*}^* - \theta_b^*) \rangle \bigg)
$$
$$
\le \exp\bigg( -(1-o(1))\frac{\mathrm{SNR}'_{z_j^*,b}}{8} \bigg).
$$

Then we have

$$
\mathbb{E}M_1 \le n\exp\bigg( -(1+o(1))\frac{\mathrm{SNR}'^2}{8} \bigg).
$$

Using the Markov's inequality we complete the proof of Lemma B.4. □

*Proof of Theorem 3.2.* By Lemmas B.2-B.4, we can obtain the result by arguments used in the proof of Theorem 2.2 and hence the proof is omitted here. □

# C  Technical Lemmas

In this section, we present and prove technical lemmas used in this paper. Lemmas C.1 and C.2 are about $\chi^2$ distributions. Appendix C.1 gives various upper bounds needed in the proofs of Appendix B. Appendix C.2 is devoted to the calculation related to $\mathrm{SNR}'$.

**Lemma C.1.** *For any $x > 0$, we have*

$$
\mathbb{P}(\chi_d^2 \ge d + 2\sqrt{dx} + 2x) \le e^{-x},
$$
$$
\mathbb{P}(\chi_d^2 \le d - 2\sqrt{dx}) \le e^{-x}.
$$

*Proof.* These results are Lemma 1 of [12]. □

**Lemma C.2.** *Let $W_i \overset{iid}{\sim} \chi_d^2$ for any $i \in [n]$ where $n, d$ are positive integers. Then we have*

$$
\mathbb{P}\bigg( \sum_{i=1}^n W_i^2 \ge 3nd^2 \bigg) \le \frac{4}{nd}.
$$

*Proof.* We have $\mathbb{E}\sum_{i=1}^n W_i^2 = nd(d+2)$ and $\mathbb{E}\sum_{i=1}^n W_i^4 = nd(d+2)(d+4)(d+6)$. Then we have $\mathrm{Var}\big(\sum_{i=1}^n W_i^2\big) = 8nd(d+2)(d+3)$. Then we obtain the desired result by Chebyshev's inequality. □

## C.1  With-High-Probability Bounds

**Lemma C.3.** *For any $z^* \in [k]^n$ and $k \in [n]$, consider independent vectors $\epsilon_j \sim \mathcal{N}(0, \Sigma_{z_j^*}^*)$ for any $j \in [n]$. Assume there exists a constant $\lambda_{\max} > 0$ such that $\|\Sigma_a^*\| \le \lambda_{\max}$ for any $a \in [k]$. Then, for*

*any constant $C' > 0$, there exists some constant $C > 0$ only depending on $C', \lambda_{\max}$ such that*

$$\max_{a \in [k]} \left\| \frac{\sum_{j=1}^n \mathbb{I}\{z_j^* = a\}\epsilon_j}{\sqrt{\sum_{j=1}^n \mathbb{I}\{z_j^* = a\}}} \right\| \leq C\sqrt{d + \log n}, \qquad (33)$$

$$\max_{a \in [k]} \frac{1}{d + \sum_{j=1}^n \mathbb{I}\{z_j^* = a\}} \left\| \sum_{j=1}^n \mathbb{I}\{z_j^* = a\}\epsilon_j\epsilon_j^T \right\| \leq C, \qquad (34)$$

$$\max_{T \subset [n]} \left\| \frac{1}{\sqrt{|T|}} \sum_{j \in T} \epsilon_j \right\| \leq C\sqrt{d + n}, \qquad (35)$$

$$\max_{a \in [k]} \max_{T \subset \{j : z_j^* = a\}} \left\| \frac{1}{\sqrt{|T|(d + \sum_{j=1}^n \mathbb{I}\{z_j^* = a\})}} \sum_{j \in T} \epsilon_j \right\| \leq C, \qquad (36)$$

*with probability at least $1 - n^{-C'}$. We have used the convention that $0/0 = 0$.*

*Proof.* Note that $\epsilon_j$ is sub-Gaussian with parameter $\lambda_{\max}$ which is a constant. The inequalities (33) and (35) are respectively Lemmas A.4, A.1 in [15]. The inequality (34) is a slight extension of Lemma A.2 in [15]. This extension follows from a standard union bound argument. The proof of (36) is identical to that of (35). $\qquad \square$

**Lemma C.4.** *Consider the same assumptions as in Lemma C.3. Assume additionally $\min_{a \in [k]} \sum_{j=1}^n \mathbb{I}\{z_j^* = a\} \geq \frac{\alpha n}{k}$ for some constant $\alpha > 0$ and $\frac{k(d + \log n)}{n} = o(1)$. Then, for any constant $C' > 0$, there exists some constant $C > 0$ only depending on $\alpha, C', \lambda_{\max}$ such that*

$$\max_{a \in [k]} \left\| \frac{1}{\sum_{j=1}^n \mathbb{I}\{z_j^* = a\}} \sum_{j=1}^n \mathbb{I}\{z_j^* = a\}\epsilon_j\epsilon_j^T - \Sigma_a^* \right\| \leq C\sqrt{\frac{k(d + \log n)}{n}}, \qquad (37)$$

*with probability at least $1 - n^{-C'}$.*

*Proof.* Note that we have $\epsilon_j = \Sigma_{z_j^*}^{*\frac{1}{2}} \eta_j$ where $\eta_j \overset{iid}{\sim} \mathcal{N}(0, I_d)$ for any $j \in [n]$. Since $\max_a \|\Sigma_a^*\| \leq \lambda_{\max}$, we have

$$\max_{a \in [k]} \left\| \frac{1}{\sum_{j=1}^n \mathbb{I}\{z_j^* = a\}} \sum_{j=1}^n \mathbb{I}\{z_j^* = a\}\epsilon_j\epsilon_j^T - \Sigma_a^* \right\| \leq \lambda_{\max} \max_{a \in [k]} \left\| \frac{1}{\sum_{j=1}^n \mathbb{I}\{z_j^* = a\}} \sum_{j=1}^n \mathbb{I}\{z_j^* = a\}\eta_j\eta_j^T - I_d \right\|.$$

Define

$$Q_a = \frac{1}{\sum_{j=1}^n \mathbb{I}\{z_j^* = a\}} \sum_{j=1}^n \mathbb{I}\{z_j^* = a\}\eta_j\eta_j^T - I_d.$$

Take $S^{d-1} = \{y \in \mathbb{R}^d : \|y\| = 1\}$ and $N_\epsilon = \{v_1, \cdots, v_{|N_\epsilon|}\}$ is an $\epsilon$-covering of $S^{d-1}$. In particular, we pick $\epsilon < \frac{1}{4}$, then $|N_\epsilon| \leq 9^d$. By the definition of the $\epsilon$-covering, we have

$$\|Q_a\| \leq \frac{1}{1 - 2\epsilon} \max_{i=1,\cdots,|N_\epsilon|} |v_i^T Q_a v_i| \leq 2 \max_{i=1,\cdots,|N_\epsilon|} |v_i^T Q_a v_i|.$$

For any $v \in N_\epsilon$,

$$v^T Q_a v = \frac{1}{\sum_{j=1}^n \mathbb{I}\{z_j^* = a\}} \sum_{j=1}^n \mathbb{I}\{z_j^* = a\}(v^T \eta_j \eta_j^T v - 1).$$

Denote $n_a = \sum_{j=1}^n \mathbb{I}\{z_j^* = a\}$. Then $\sum_{j=1}^n \mathbb{I}\{z_j^* = a\} v^T \eta_j \eta_j^T v \sim \chi_{n_a}^2$. Using Lemma C.1, we have

$$\mathbb{P}(\max_{a \in [k]} \|Q_a\| \geq t) \leq \sum_{a=1}^k \mathbb{P}(\|Q_a\| \geq t)$$

$$\leq \sum_{a=1}^k \sum_{i=1}^{|N_\epsilon|} \mathbb{P}(|v_i^T Q_a v_i| \geq t/2)$$

$$\leq \sum_{a=1}^k 2 \exp\left\{-\frac{n_a}{8} \min\{t, t^2\} + d \log 9\right\}.$$

Since $\frac{k(d+\log n)}{n} = o(1)$ and $n_a \geq \alpha n/k$ where $\alpha$ is a constant, we can take $t = C'' \sqrt{\frac{k(d+\log n)}{n}}$ for some large constant $C''$ and the proof is complete. $\qquad\square$

**Lemma C.5.** *Consider the same assumptions as in Lemma C.3. Then, for any $s = o(n)$ and for any constant $C' > 0$, there exists some constant $C > 0$ only depending on $C', \lambda_{\max}$ such that*

$$\max_{T \subset [n]:|T| \leq s} \frac{1}{|T| \log \frac{n}{|T|} + \min\{1, \sqrt{|T|}\}d} \left\| \sum_{j \in T} \epsilon_j \epsilon_j^T \right\| \leq C, \tag{38}$$

*with probability at least $1 - n^{-C'}$. We have used the convention that $0/0 = 0$.*

*Proof.* Consider any $a \in [s]$ and a fixed $T \subset [n]$ such that $|T| = a$. Similar to the proof of Lemma C.4, we can take $S^{d-1} = \{y \in \mathbb{R}^d : \|y\| = 1\}$ and its $\epsilon$-covering $N_\epsilon$ with $\epsilon < \frac{1}{4}$ and $|N_\epsilon| \leq 9^d$. Then we have

$$\|\sum_{j \in T} \epsilon_j \epsilon_j^T \| = \sup_{\|w\|=1} \sum_{j \in T} (w^T \epsilon_j)^2 \leq 2 \max_{w \in N_\epsilon} \sum_{j \in T} (w^T \epsilon_j)^2.$$

Note that $w^T \epsilon_j / \sqrt{\lambda_{\max}}$ is a sub-Gaussian random variable with parameter 1. By the tail probability result for quadratic forms of sub-Gaussian random vectors [10], for any fixed $w \in N_\epsilon$, we have

$$\mathbb{P}\left(\sum_{j \in T} (w^T \epsilon_j)^2 \geq \lambda_{\max}\left(a + 2\sqrt{at} + 2t\right)\right) \leq \exp(-t).$$

Since $a = o(n)$, there exists a constant $C_0$ such that $2a \leq C_0 a \log \frac{n}{a}$. We can take $t = \tilde{C}(a \log \frac{n}{a} + d)$ with $\tilde{C} = \frac{C}{16} - \frac{C_0}{4}$, then $a + 2\sqrt{at} + 2t \leq \frac{C}{4}(a \log \frac{n}{a} + d)$. Thus,

$$\mathbb{P}\left(\sum_{j \in T} (w^T \epsilon_j)^2 \geq \frac{C}{4}(a \log \frac{n}{a} + d)\right) \leq \exp\left(-\tilde{C}(a \log \frac{n}{a} + d)\right).$$

Hence, we have

$$\mathbb{P}\left(\|\sum_{j \in T} \epsilon_j \epsilon_j^T\| \geq \frac{C}{2}(a \log \frac{n}{a} + d)\right) \leq 9^d \exp\left(-\tilde{C}(a \log \frac{n}{a} + d)\right).$$

As a result,

$$\mathbb{P}\left\{\max_{T \subset [n], 1 \leq |T| \leq s} \frac{1}{|T| \log \frac{n}{|T|} + d} \|\sum_{j \in T} \epsilon_j \epsilon_j^T\| \geq C\right\} \leq \sum_{a=1}^s \mathbb{P}\left\{\max_{|T|=a} \|\sum_{j \in T} \epsilon_j \epsilon_j^T\| \geq C(a \log \frac{n}{a} + d)\right\}$$

$$\leq \sum_{a=1}^s \binom{n}{a} \max_{|T|=a} \mathbb{P}\left\{\|\sum_{j \in T} \epsilon_j \epsilon_j^T\| \geq C(a \log \frac{n}{a} + d)\right\}$$

$$\leq \sum_{a=1}^s \binom{n}{a} 9^d \exp\left(-\tilde{C}(a \log \frac{n}{a} + d)\right).$$

Since $a \log \frac{n}{a}$ is an increasing function when $a \in [1, s]$ and $a \log \frac{n}{a} \geq \log n \geq \log s$, a choice of $\tilde{C} = 3 + C'$, that is $C = 16C' + 4C_0 + 48$, can yield the desired result.

Finally, to allow $|T| = 0$, we note that $d \leq \min\{1, \sqrt{|T|}\}d$. The proof is complete. $\qquad\square$

**Lemma C.6.** *For any $z^* \in [k]^n$ and $k \in [n]$, assume $\min_{a \in [k]} \sum_{j=1}^n \mathbb{I}\{z_j^* = a\} \geq \frac{\alpha n}{k}$ and $\ell(z, z^*) = o(\frac{n\Delta^2}{k})$, then*

$$\max_{a \in [k]} \frac{\sum_{j=1}^n \mathbb{I}\{z_j^* = a\}}{\sum_{j=1}^n \mathbb{I}\{z_j = a\}} \leq 2. \tag{39}$$

*Proof.* For any $z \in [k]^n$ such that $\ell(z, z^*) = o(n)$ and any $a \in [k]$, we have

$$\sum_{j=1}^n \mathbb{I}\{z_j = a\} \geq \sum_{j=1}^n \mathbb{I}\{z_j^* = a\} - \sum_{j=1}^n \mathbb{I}\{z_j \neq z_j^*\}$$

$$\geq \sum_{j=1}^n \mathbb{I}\{z_j^* = a\} - \frac{\ell(z, z^*)}{\Delta^2}$$

$$\geq \frac{\alpha n}{2k}, \tag{40}$$

which implies

$$\frac{\sum_{j=1}^n \mathbb{I}\{z_j^* = a\}}{\sum_{j=1}^n \mathbb{I}\{z_j = a\}} \leq \frac{\sum_{j=1}^n \mathbb{I}\{z_j = a\} + \sum_{j=1}^n \mathbb{I}\{z_j \neq z_j^*\}}{\sum_{j=1}^n \mathbb{I}\{z_j = a\}}$$

$$\leq 1 + \frac{\alpha n / 2k}{\sum_{j=1}^n \mathbb{I}\{z_j = a\}}$$

$$\leq 2.$$

Thus, we obtain (39). $\qquad\square$

In the following lemma, we are going to analyze estimation errors of the centers and covariance matrices under the anisotropic GMMs. For any $z \in [k]^n$ and for any $z \in [k]$, recall the definitions

$$\hat{\theta}_a(z) = \frac{\sum_{j \in [n]} Y_j \mathbb{I}\{z_j = a\}}{\sum_{j \in [n]} \mathbb{I}\{z_j = a\}}, \forall a \in [k]$$

$$\hat{\Sigma}_a(z) = \frac{\sum_{j \in [n]} (Y_j - \hat{\theta}_a(z))(Y_j - \hat{\theta}_a(z))^T \mathbb{I}\{z_j = a\}}{\sum_{j \in [n]} \mathbb{I}\{z_j = a\}}, \forall a \in [k].$$

**Lemma C.7.** *For any $z^* \in [k]^n$ and $k \in [n]$, consider independent vectors $Y_j = \theta_{z_j^*}^* + \epsilon_j$ where $\epsilon_j \sim \mathcal{N}(0, \Sigma_{z_j^*}^*)$ for any $j \in [n]$. Assume there exist constants $\lambda_{\min}, \lambda_{\max} > 0$ such that $\lambda_{\min} \leq \lambda_1(\Sigma_a^*) \leq \lambda_d(\Sigma_a^*) \leq \lambda_{\max}$ for any $a \in [k]$, and a constant $\alpha > 0$ such that $\min_{a \in [k]} \sum_{j=1}^n \mathbb{I}\{z_j^* = a\} \geq \frac{\alpha n}{k}$. Assume $\frac{k(d + \log n)}{n} = o(1)$ and $\frac{\Delta}{k} \to \infty$. Assume (33)-(38) hold. Then for any $\tau = o(n)$ and for any constant $C'' > 0$, there exists some constant $C > 0$ only depending on $\alpha, \lambda_{\max}, C'$ such that*

$$\max_{a \in [k]} \left\| \hat{\theta}_a(z^*) - \theta_a^* \right\| \leq C \sqrt{\frac{k(d + \log n)}{n}}, \tag{41}$$

$$\max_{a \in [k]} \left\| \hat{\theta}_a(z) - \hat{\theta}_a(z^*) \right\| \leq C \left( \frac{k}{n\Delta} \ell(z, z^*) + \frac{k\sqrt{d + n}}{n\Delta} \sqrt{\ell(z, z^*)} \right), \tag{42}$$

$$\max_{a \in [k]} \left\| \hat{\Sigma}_a(z^*) - \Sigma_a^* \right\| \leq C \sqrt{\frac{k(d + \log n)}{n}}, \tag{43}$$

$$\max_{a \in [k]} \left\| \hat{\Sigma}_a(z) - \hat{\Sigma}_a(z^*) \right\| \leq C \left( \frac{k}{n} \ell(z, z^*) + \frac{k\sqrt{n\ell(z, z^*)}}{n\Delta} + \frac{kd}{n\Delta} \sqrt{\ell(z, z^*)} \right), \tag{44}$$

*for all $z$ such that $\ell(z, z^*) \leq \tau$.*

*Proof.* Using (33) we obtain (41). By the same argument of (118) in [8], we can obtain (42). By (33) and (37) and (41), we can obtain (43). In the remaining proof, we will establish (53).

Since $\frac{k(d+\log n)}{n} = o(1)$, we have $\|\hat{\Sigma}_a(z^*)\| \preceq 1$ for any $a \in [k]$. The difference $\hat{\Sigma}_a(z) - \hat{\Sigma}_a(z^*)$ will be decomposed into several terms. We notice that

$$\left\| \hat{\Sigma}_a(z) - \hat{\Sigma}_a(z^*) \right\| \le S_1 + S_2, \tag{45}$$

where

$$S_1 = \left\| \frac{1}{\sum \mathbb{I}\{z_j = a\}} \sum_{j=1}^n \mathbb{I}\{z_j = a\} \left( (Y_j - \hat{\theta}_a(z))(Y_j - \hat{\theta}_a(z))^T - (Y_j - \hat{\theta}_a(z^*))(Y_j - \hat{\theta}_a(z^*))^T \right) \right\|,$$

and

$$S_2 = \left\| \left( \frac{1}{\sum \mathbb{I}\{z_j = a\}} - \frac{1}{\sum \mathbb{I}\{z_j^* = a\}} \right) \sum_{j=1}^n \mathbb{I}\{z_j^* = a\}(Y_j - \hat{\theta}_a(z^*))(Y_j - \hat{\theta}_a(z^*))^T \right\|.$$

Also, we notice that

$$S_1 \le L_1 + L_2 + L_3, \tag{46}$$

where

$$L_1 = \left\| \frac{1}{\sum_{j=1}^n \mathbb{I}\{z_j = a\}} \sum_{j=1}^n \mathbb{I}\{z_j = z_j^* = a\} \left( (Y_j - \hat{\theta}_a(z))(Y_j - \hat{\theta}_a(z))^T - (Y_j - \hat{\theta}_a(z^*))(Y_j - \hat{\theta}_a(z^*))^T \right) \right\|,$$

$$L_2 = \left\| \frac{1}{\sum_{j=1}^n \mathbb{I}\{z_j = a\}} \sum_{j=1}^n \mathbb{I}\{z_j = a, z_j^* \neq a\}(Y_j - \hat{\theta}_a(z))(Y_j - \hat{\theta}_a(z))^T \right\|,$$

$$L_3 = \left\| \frac{1}{\sum_{j=1}^n \mathbb{I}\{z_j = a\}} \sum_{j=1}^n \mathbb{I}\{z_j \neq a, z_j^* = a\}(Y_j - \hat{\theta}_a(z^*))(Y_j - \hat{\theta}_a(z^*))^T \right\|.$$

For $L_1$, we have

$$L_1 \le \left\| \frac{1}{\sum_{j=1}^n \mathbb{I}\{z_j = a\}} \sum_{j=1}^n \mathbb{I}\{z_j = z_j^* = a\}(\hat{\theta}_a(z) - \hat{\theta}_a(z^*))(\hat{\theta}_a(z) - \hat{\theta}_a(z^*))^T \right\|$$

$$+ 2 \left\| \frac{1}{\sum_{j=1}^n \mathbb{I}\{z_j = a\}} \sum_{j=1}^n \mathbb{I}\{z_j = z_j^* = a\}(Y_j - \hat{\theta}_a(z^*))(\hat{\theta}_a(z) - \hat{\theta}_a(z^*))^T \right\|$$

$$\preceq \left\| \hat{\theta}_a(z) - \hat{\theta}_a(z^*) \right\|^2 \frac{\sum_{j=1}^n \mathbb{I}\{z_j^* = a\}}{\sum_{j=1}^n \mathbb{I}\{z_j = a\}} + \left\| \theta_a^* - \hat{\theta}_a(z^*) \right\| \left\| \hat{\theta}_a(z) - \hat{\theta}_a(z^*) \right\| \frac{\sum_{j=1}^n \mathbb{I}\{z_j^* = a\}}{\sum_{j=1}^n \mathbb{I}\{z_j = a\}}$$

$$+ \left\| \hat{\theta}_a(z) - \hat{\theta}_a(z^*) \right\| \left\| \frac{1}{\sum_{j=1}^n \mathbb{I}\{z_j = a\}} \sum_{j=1}^n \mathbb{I}\{z_j = z_j^* = a\}\epsilon_j \right\|. \tag{47}$$

By (36), (39), (40), we have uniformly for any $a \in [k]$,

$$\left\| \frac{1}{\sum_{j=1}^n \mathbb{I}\{z_j = a\}} \sum_{j=1}^n \mathbb{I}\{z_j = z_j^* = a\}\epsilon_j \right\| \preceq \frac{\sqrt{\sum_{j=1}^n \mathbb{I}\{z_j = z_j^* = a\}}}{\sum_{j=1}^n \mathbb{I}\{z_j = a\}} \sqrt{d + \sum_{j=1}^n \mathbb{I}\{z_j^* = a\}}$$

$$\preceq 1. \tag{48}$$

Since $\max_{a \in [k]} \left\| \hat{\theta}_a(z^*) - \theta_a^* \right\| = o(1)$, by (39), (42), (41), (47), and (48), we have uniformly for any $a \in [k]$,

$$L_1 \preceq \left\| \hat{\theta}_a(z) - \hat{\theta}_a(z^*) \right\| \preceq \frac{k}{n\Delta} \ell(z, z^*) + \frac{k\sqrt{d+n}}{n\Delta} \sqrt{\ell(z, z^*)}. \tag{49}$$

To bound $L_2$, we first give the following simple fact. For any positive integer $m$ and any $\{u_j\}_{j\in[m]}, \{v_j\}_{j\in[m]} \in \mathbb{R}^d$, we have $\|\sum_{j\in[m]}(u_j + v_j)(u_j + v_j)^T\| \leq 2\|\sum_{j\in[m]} u_j u_j^T\| + 2\|\sum_{j\in[m]} v_j v_j^T\|$. Hence, for $L_2$, we have the following decomposition

$$L_2 \leq 2R_1 + 2R_2, \tag{50}$$

where

$$R_1 = \left\| \frac{1}{\sum_{j=1}^n \mathbb{I}\{z_j = a\}} \sum_{j=1}^n \mathbb{I}\{z_j = a, z_j^* \neq a\}(Y_j - \theta_a^*)(Y_j - \theta_a^*)^T \right\|,$$

$$R_2 = \left\| \frac{1}{\sum_{j=1}^n \mathbb{I}\{z_j = a\}} \sum_{j=1}^n \mathbb{I}\{z_j = a, z_j^* \neq a\}(\theta_a^* - \hat{\theta}_a(z))(\theta_a^* - \hat{\theta}_a(z))^T \right\|.$$

Since $\max_{a\in[k]} \sum_{j=1}^n \mathbb{I}\{z_j = a, z_j^* \neq a\} \leq \frac{\ell(z,z^*)}{\Delta^2}$, we have

$$R_2 \leq \left\| \theta_a^* - \hat{\theta}_a(z) \right\|^2 \frac{\sum_{j=1}^n \mathbb{I}\{z_j = a, z_j^* \neq a\}}{\sum_{j=1}^n \mathbb{I}\{z_j = a\}}$$

$$\preceq \left( \left\| \hat{\theta}_a(z) - \hat{\theta}_a(z^*) \right\|^2 + \left\| \hat{\theta}_a(z^*) - \theta_a^* \right\|^2 \right) \frac{k\ell(z,z^*)}{n\Delta^2}. \tag{51}$$

By (38) and the fact that $\max_{a\in[k]} \sum_{j=1}^n \mathbb{I}\{z_i = a, z_i^* \neq a\} \leq \frac{\ell(z,z^*)}{\Delta^2}$, we also have

$$R_1 \leq 2 \left\| \frac{1}{\sum_{j=1}^n \mathbb{I}\{z_j = a\}} \sum_{j=1}^n \mathbb{I}\{z_j = a, z_j^* \neq a\}(\theta_{z_j^*}^* - \theta_{z_j}^*)(\theta_{z_j^*}^* - \theta_{z_j}^*)^T \right\|$$

$$+ 2 \left\| \frac{1}{\sum_{j=1}^n \mathbb{I}\{z_j = a\}} \sum_{j=1}^n \mathbb{I}\{z_j = a, z_j^* \neq a\}\epsilon_j \epsilon_j^T \right\|$$

$$\leq 2 \frac{\sum_{j=1}^n \mathbb{I}\{z_j = a, z_j^* \neq a\}\|\theta_{z_j^*}^* - \theta_{z_j}^*\|^2}{\sum_{j=1}^n \mathbb{I}\{z_j = a\}} + 2 \left\| \frac{1}{\sum_{j=1}^n \mathbb{I}\{z_j = a\}} \sum_{j=1}^n \mathbb{I}\{z_j = a, z_j^* \neq a\}\epsilon_j \epsilon_j^T \right\|$$

$$\preceq \frac{k\ell(z,z^*)}{n} + \frac{\frac{\ell(z,z^*)}{\Delta^2} \log \frac{n\Delta^2}{\ell(z,z^*)} + d\sqrt{\frac{\ell(z,z^*)}{\Delta^2}}}{n/k}.$$

We are going to simplify the above bounds for $R_1, R_2$. Under the assumption that $\frac{k(d+\log n)}{n} = o(1)$, $\Delta/k \to \infty$, and $\ell(z,z^*) \leq \tau = o(n)$, we have $\max_{a\in[k]} \|\hat{\theta}_a(z) - \hat{\theta}_a(z^*)\| = o(1)$, $\max_{a\in[k]} \|\hat{\theta}_a(z^*) - \theta_a^*\| = o(1)$, and $\frac{k\ell(z,z^*)}{n\Delta^2} = o(1)$. Hence $R_2 \preceq \frac{k\ell(z,z^*)}{n\Delta^2}$. Also we have

$$\frac{k\ell(z,z^*)}{n\Delta^2} \log \frac{n\Delta^2}{\ell(z,z^*)} = \frac{k\sqrt{\ell(z,z^*)}}{n\Delta} \sqrt{\frac{\ell(z,z^*)}{\Delta^2} \left( \log \frac{n\Delta^2}{\ell(z,z^*)} \right)^2} \leq \frac{k\sqrt{n\ell(z,z^*)}}{n\Delta}.$$

where in the last inequality, we use the fact that $x(\log(n/x))^2$ is an increasing function of $x$ when $0 < x = o(n)$. Then,

$$L_2 \preceq \frac{k\sqrt{n\ell(z,z^*)}}{n\Delta} + \frac{k}{n}\ell(z,z^*) + \frac{kd}{n\Delta}\sqrt{\ell(z,z^*)}.$$

Since $L_3$ is similar to $L_2$, by (46) we have uniformly for any $a \in [k]$

$$S_1 \preceq \frac{k\sqrt{n\ell(z,z^*)}}{n\Delta} + \frac{k}{n}\ell(z,z^*) + \frac{kd}{n\Delta}\sqrt{\ell(z,z^*)}. \tag{52}$$

To bound $S_2$, by (70) in [8], we have uniformly for any $a \in [k]$,

$$S_2 = \frac{\left| \sum_{j=1}^n \mathbb{I}\{z_j^* = a\} - \sum_{j=1}^n \mathbb{I}\{z_j = a\} \right|}{\sum_{j=1}^n \mathbb{I}\{z_j = a\}} \left\| \hat{\Sigma}_a(z^*) \right\|^2 \preceq \frac{k}{n}\frac{\ell(z,z^*)}{\Delta^2},$$

where we use (43). Since $\frac{k}{n}\frac{\ell(z,z^*)}{\Delta^2} \preceq \frac{k\sqrt{n\ell(z,z^*)}}{n\Delta}$, by (45) and the facts that $\ell(z,z^*) \leq \tau = o(n)$ we have

$$\max_{a\in[k]}\left\|\hat{\Sigma}_a(z) - \hat{\Sigma}_a(z^*)\right\| \preceq \frac{k\sqrt{n\ell(z,z^*)}}{n\Delta} + \frac{k}{n}\ell(z,z^*) + \frac{kd}{n\Delta}\sqrt{\ell(z,z^*)}.$$

$\square$

**Lemma C.8.** *Under the same assumption as in Lemma C.7, if additionally we assume $kd = O(\sqrt{n})$ and $\tau = o(n/k)$, there exists some constant $C > 0$ only depending on $\alpha, \lambda_{\min}, \lambda_{\max}, C'$ such that*

$$\max_{a\in[k]}\left\|(\hat{\Sigma}_a(z))^{-1} - (\hat{\Sigma}_a(z^*))^{-1}\right\| \leq C\left(\frac{k}{n}\ell(z,z^*) + \frac{k\sqrt{n\ell(z,z^*)}}{n\Delta} + \frac{kd}{n\Delta}\sqrt{\ell(z,z^*)}\right). \quad (53)$$

*Proof.* By (43) we have $\max_{a\in[k]}\|\hat{\Sigma}_a(z^*)\|, \max_{a\in[k]}\|(\hat{\Sigma}_a(z^*))^{-1}\| \preceq 1$. By (44) we also have $\max_{a\in[k]}\|\hat{\Sigma}_a(z)\|, \max_{a\in[k]}\|(\hat{\Sigma}_a(z))^{-1}\| \preceq 1$. Hence,

$$\max_{a\in[k]}\left\|(\hat{\Sigma}_a(z))^{-1} - (\hat{\Sigma}_a(z^*))^{-1}\right\| \leq \max_{a\in[k]}\left\|(\hat{\Sigma}_a(z^*))^{-1}\right\|\left\|\hat{\Sigma}_a(z) - \hat{\Sigma}_a(z^*)\right\|\left\|(\hat{\Sigma}_a(z))^{-1}\right\|$$

$$\preceq \frac{k\sqrt{n\ell(z,z^*)}}{n\Delta} + \frac{k}{n}\ell(z,z^*) + \frac{kd}{n\Delta}\sqrt{\ell(z,z^*)}. \quad (54)$$

$\square$

### C.2 Calculation Related to SNR$'$

In the following lemmas, we study properties of $\{\text{SNR}'_{a,b}\}_{a\neq b}$. Consider any pair $a \neq b \in [k]$. Let $\eta \sim \mathcal{N}(0, I_d)$ and $\Xi_{a,b} = \theta_a^* - \theta_b^*$. Define

$$B_{a,b}(\delta) = \left\{x \in \mathbb{R}^d : x^T\Sigma_a^{*\frac{1}{2}}(\Sigma_b^*)^{-1}\Xi_{a,b} + \frac{1}{2}x^T\left(\Sigma_a^{*\frac{1}{2}}(\Sigma_b^*)^{-1}\Sigma_a^{*\frac{1}{2}} - I_d\right)x\right.$$

$$\left. \leq -\frac{1-\delta}{2}\Xi_{a,b}^T(\Sigma_b^*)^{-1}\Xi_{a,b} + \frac{1}{2}\log|\Sigma_a^*| - \frac{1}{2}\log|\Sigma_b^*|\right\}$$

$$= \left\{x \in \mathbb{R}^d : \|x\|^2 \geq \left(x - (\Sigma_a^*)^{-\frac{1}{2}}\Xi_{b,a}\right)^T\left((\Sigma_a^*)^{-\frac{1}{2}}\Sigma_b^*(\Sigma_a^*)^{-\frac{1}{2}}\right)^{-1}\left(x - (\Sigma_a^*)^{-\frac{1}{2}}\Xi_{b,a}\right)\right.$$

$$\left. + \log|(\Sigma_a^*)^{-\frac{1}{2}}\Sigma_b^*(\Sigma_a^*)^{-\frac{1}{2}}| - \delta\Xi_{b,a}^T(\Sigma_b^*)^{-1}\Xi_{b,a}\right\},$$

for any $\delta \in \mathbb{R}$. Then $B_{a,b}(\delta) \subset B_{a,b}(\delta')$ for any $\delta' \leq \delta$. In addition, we define

$$\text{SNR}'_{a,b}(\delta) = \min_{x\in B_{a,b}(\delta)} 2\|x\|,$$

$$\text{and } P_{a,b}(\delta) = \mathbb{P}(\eta \in B_{a,b}(\delta)).$$

Recall the definitions of $B_{a,b}$ and $\text{SNR}'_{a,b}$ in Section 3. Then it is a special case of $B_{a,b}(\delta)$ and $\text{SNR}'_{a,b}(\delta)$ with $\delta = 0$. That is, we have $B_{a,b} = B_{a,b}(0)$ and $\text{SNR}'_{a,b} = \text{SNR}'_{a,b}(0)$.

To understand these quantities, we first study a canonical setting that can be later applied to establish Lemma C.10.

**Lemma C.9.** *Consider any $\theta \in \mathbb{R}^d \setminus \{0\}$ and any $\Sigma \in \mathbb{R}^{d\times d}$ that is positive semi-definite. Let $\lambda_{\max}, \lambda_{\min} > 0$ be the largest and smallest eigenvalue of $\Sigma$, respectively. For any $t \in \mathbb{R}$, define*

$$D(t) = \{x \in \mathbb{R}^d : \|x\|^2 \geq (x - \theta)^T\Sigma^{-1}(x - \theta) + t\},$$

*and $s(t) = \min_{x\in D(t)}\|x\|$. Then the following hold:*

- *Under the assumption that $-\|\theta\|^2/(8\lambda_{\max}) < t < \|\theta\|^2/8$, we have*

$$\|\theta\|/(\max\{2, 2\sqrt{2\lambda_{\max}}\}) < s(t) < (1 - \min\{\sqrt{\lambda_{\min}/8}, 1/2\})\|\theta\|.$$

- *If $t'$ also satisfies $-\|\theta\|^2/\lambda_{\max} < t' < \|\theta\|^2/8$, we have*

$$|s(t') - s(t)| \le \lambda_{\max} \frac{t' - t}{2\min\{\sqrt{\lambda_{\min}/8}, 1/2\}\|\theta\|}.$$

- *If $\theta$ is further assumed to satisfy $\min\{\sqrt{\lambda_{\min}/8}, 1/2\}\|\theta\| \ge 2\lambda_{\max}$ and $\|\theta\| \ge \frac{\lambda_{\min}}{32}\min\{\sqrt{\lambda_{\min}/8}, 1/2\}$, there exists a $d$-dimensional ball $H(t) \in \mathbb{R}^d$ with radius $(\lambda_{\min}/8)\min\{\sqrt{\lambda_{\min}/8}, 1/2\}$ such that $H(t) \subset D(t)$ and $\|x\| \le (\lambda_{\min}/8)\min\{\sqrt{\lambda_{\min}/8}, 1/2\} + \lambda_{\max} + s(t)$ for all $x \in H(t)$.*

*Proof.* First, we check whether each of the following two points is contained in $D(t)$ or not, under the assumption $-\|\theta\|^2/\lambda_{\max} < t < \|\theta\|^2$.

- When $x = 0$, we have $(x-\theta)^T\Sigma^{-1}(x-\theta) + t \ge \|\theta\|^2/\lambda_{\max} + t > 0$. Hence, $0 \notin D(t)$.

- When $x = \theta$, we have $\|\theta\|^2 > t$. Hence, $\theta \in D(t)$.

As a result, $D(t)$ is non-empty, $s(t)$ is well defined, and $0 < s(t) < \|\theta\|^2$. Next, we consider a few more points to sharpen upper and lower bounds on $s(t)$ under the assumption $-\|\theta\|^2/(8\lambda_{\max}) < t < \|\theta\|^2/8$.

- For any $x$ that satisfies $\|x\| \le \|\theta\|/(\max\{2, 2\sqrt{2\lambda_{\max}}\})$, we have $\|x-\theta\| \ge \|\theta\|/2$ and consequently, $(x-\theta)^T\Sigma^{-1}(x-\theta) + t \ge (\|\theta\|/2)^2/\lambda_{\max} + t = \|\theta\|^2/(4\lambda_{\max}) + t$. Under the assumption that $t > -\|\theta\|^2/(8\lambda_{\max})$, we can verify that $\|\theta\|^2/(4\lambda_{\max}) + t > \|\theta\|^2/(8\lambda_{\max}) \ge (\|\theta\|/\max\{2, 2\sqrt{2\lambda_{\max}}\})^2 \ge \|x\|^2$. Hence, such $x \notin D(t)$.

- When $x = (1 - \min\{\sqrt{\lambda_{\min}/8}, 1/2\})\theta$, we have $\|x\| \ge \|\theta\|/2$ and $(x-\theta)^T\Sigma^{-1}(x-\theta) + t \le \|\theta\|^2(\min\{\sqrt{\lambda_{\min}/8}, 1/2\})^2/\lambda_{\min} + t = \|\theta\|^2/\max\{8, 4\lambda_{\min}\} + t$. Under the assumption that $t < \|\theta\|^2/8$, we have $\|\theta\|^2/\max\{8, 4\lambda_{\min}\} + t < \|\theta\|^2/8 + t \le \|\theta\|^2/4 \le \|x\|^2$. Hence, such $x \in D(t)$.

As a result, we have $\|\theta\|/(\max\{2, 2\sqrt{2\lambda_{\max}}\}) < s(t) < (1 - \min\{\sqrt{\lambda_{\min}/8}, 1/2\})\|\theta\|$.

Define a ball $\mathcal{S}(r) = \{x \in \mathbb{R}^d : \|x\|^2 \le r^2\}$ and define $\mathcal{S}_2(r; t) = \{x \in \mathbb{R}^d : (x-\theta)^T\Sigma^{-1}(x-\theta) \le r^2 - t\}$ to be the part of $\mathbb{R}^d$ that is inside the corresponding ellipsoid. Then we have $s(t) = \min\{r \ge 0 : \mathcal{S}_1(r) \cap \mathcal{S}_2(r; t) \ne \emptyset\}$. By the definition and bounds of $s(t)$ and the convexity of $\mathcal{S}_1(s(t))$ and $\mathcal{S}_2(s(t); t)$, we must have $|\mathcal{S}_1(s(t)) \cap \mathcal{S}_2(s(t); t)| = 1$, meaning that $\mathcal{S}_1(s(t))$ and $\mathcal{S}_2(s(t); t)$ touch each other externally at one point. This implies $s(t)$ can be obtained by the following process: We let $\mathcal{S}_1(r)$ and $\mathcal{S}_2(r)$ grow by increasing $r$, starting from 0. The first time they touch each other, we stop and the value of $r$ is exactly $s(t)$.

Denote $y(t) \in \mathbb{R}$ such that $\{y(t)\} = \mathcal{S}_1(s(t)) \cap \mathcal{S}_2(s(t); t)$. Then we must have $y(t) \in \bar{\mathcal{S}}_2(s(t); t)$. By the first conclusion, we have

$$\|y(t) - \theta\| \ge \|\theta\| - \|y(t)\| = \|\theta\| - s(t) \ge \min\{\sqrt{\lambda_{\min}/8}, 1/2\}\|\theta\|$$
$$\|y(t) - \theta\| \le \|\theta\| + \|y(t)\| = \|\theta\| + s(t) \le 2\|\theta\|.$$

In addition, we have

$$s^2(t) - t = (y(t)-\theta)^T\Sigma^{-1}(y(t)-\theta) \ge \|y(t)-\theta\|^2/\lambda_{\max} \ge (\min\{\sqrt{\lambda_{\min}/8}, 1/2\}\|\theta\|)^2/\lambda_{\max}.$$

Now we are going to prove the second conclusion of the lemma. Without loss of generality, assume $t \le t'$. Then we have $D(t) \supset D(t')$ and $s(t) \le s(t')$. We are going to establish a lower bound for $s(t)$. First by definition of $s(t')$, we have $|\mathcal{S}_1(s(t')) \cap \mathcal{S}_2(s(t'); t')| = 1$. Since $t \le t'$, we have $\mathcal{S}_2(s(t'); t) \supset \mathcal{S}_2(s(t'); t')$. We have $\mathcal{S}_1(s(t')) \cap \mathcal{S}_2(s(t'); t) \ne \emptyset$. From here we can also see

$s(t) \le s(t')$. Now consider any $x \in \bar{\mathcal{S}}_2(s(t'); t)$. It satisfies $(x - \theta)^T \Sigma^{-1}(x - \theta) = s^2(t') - t$. Then we have

$$\left( \sqrt{\frac{s^2(t') - t'}{s^2(t') - t}}(x - \theta) \right) \Sigma^{-1} \left( \sqrt{\frac{s^2(t') - t'}{s^2(t') - t}}(x - \theta) \right) = s^2(t') - t',$$

meaning that $\theta + \sqrt{\frac{s^2(t')-t'}{s^2(t')-t}}(x - \theta) \in \bar{\mathcal{S}}_2(s(t'); t')$. Hence,

$$
\begin{aligned}
\|x\| &\ge \left\| \theta + \sqrt{\frac{s^2(t') - t'}{s^2(t') - t}}(x - \theta) \right\| - \left\| x - \left( \theta + \sqrt{\frac{s^2(t') - t'}{s^2(t') - t}}(x - \theta) \right) \right\| \\
&= \left\| \theta + \sqrt{\frac{s^2(t') - t'}{s^2(t') - t}}(x - \theta) \right\| - \left( 1 - \sqrt{\frac{s^2(t') - t'}{s^2(t') - t}} \right) \|x - \theta\| \\
&\ge \min_{y \in \bar{\mathcal{S}}_2(s(t');t')} \|y\| - \left( 1 - \sqrt{\frac{s^2(t') - t'}{s^2(t') - t}} \right) \max_{y \in \bar{\mathcal{S}}_2(s(t');t)} \|x - \theta\|.
\end{aligned}
$$

Since $|\mathcal{S}_1(s(t')) \cap \bar{\mathcal{S}}_2(s(t'); t')| = 1$, we have $\min_{y \in \bar{\mathcal{S}}_2(s(t');t')} \|y\| = s(t')$. Since $(x - \theta)^T \Sigma^{-1}(x - \theta) \ge \lambda_{\max}^{-1} \|x - \theta\|^2$, we have $\|x - \theta\|^2 \le \lambda_{\max}(s^2(t') - t)$. Hence,

$$
\begin{aligned}
\|x\| &\ge s(t') - \left( 1 - \sqrt{\frac{s^2(t') - t'}{s^2(t') - t}} \right) \sqrt{\lambda_{\max}(s^2(t') - t)} \\
&\ge s(t') - \sqrt{\lambda_{\max}} \left( \sqrt{s^2(t') - t} - \sqrt{s^2(t') - t'} \right) \\
&= s(t') - \sqrt{\lambda_{\max}} \frac{t' - t}{\sqrt{s^2(t') - t} + \sqrt{s^2(t') - t'}} \\
&\ge s(t') - \sqrt{\lambda_{\max}} \frac{t' - t}{2\sqrt{s^2(t') - t'}}.
\end{aligned}
$$

As a result, for any $r < s(t') - \sqrt{\lambda_{\max}} \frac{t'-t}{2\sqrt{s^2(t')-t'}}$, we have $\mathcal{S}_1(r) \cap \mathcal{S}_2(s(t'); t) = \emptyset$ and consequently $\mathcal{S}_1(r) \cap \mathcal{S}_2(r; t) = \emptyset$. As a result, $s(t) \ge s(t') - \sqrt{\lambda_{\max}} \frac{t'-t}{2\sqrt{s^2(t')-t'}}$. Since we have shown $s^2(t') - t' \ge (\min\{\sqrt{\lambda_{\min}/8}, 1/2\} \|\theta\|)^2 / \lambda_{\max}$, we have $s(t) \ge s(t') - \lambda_{\max} \frac{t'-t}{2\min\{\sqrt{\lambda_{\min}/8}, 1/2\}\|\theta\|}$.

For the third conclusion of the lemma, recall the definition of $y(t)$. Under the assumption that $\min\{\sqrt{\lambda_{\min}/8}, 1/2\} \|\theta\| \ge 2\lambda_{\max}$, we have $\|y(t) - \theta\| - \lambda_{\max} > 0$ and $\lambda_{\max}/\|y(t) - \theta\| \le 1/2$. Denote $y'(t) = \theta + \frac{y(t)-\theta}{\|y(t)-\theta\|}(\|y(t) - \theta\| - \lambda_{\max})$. Then

$$
\begin{aligned}
\|y'(t) - \theta\| &\le \|y(t) - \theta\| \le 2\|\theta\|, \\
\|y'(t) - \theta\| &= (1 - \lambda_{\max}/\|y(t) - \theta\|)\|y(t) - \theta\|, \\
\|y(t) - y'(t)\| &= \left\| \frac{y(t) - \theta}{\|y(t) - \theta\|}\lambda_{\max} \right\| = \lambda.
\end{aligned}
$$

Consequently,

$$
(y'(t) - \theta)^T \Sigma^{-1}(y'(t) - \theta) - (y(t) - \theta)^T \Sigma^{-1}(y(t) - \theta)
$$

$$
= \left(1 - \frac{\lambda_{\max}}{\|y(t) - \theta\|}\right)^2 (y(t) - \theta)^T \Sigma^{-1}(y(t) - \theta) - (y(t) - \theta)^T \Sigma^{-1}(y(t) - \theta)
$$

$$
= -\frac{\lambda_{\max}}{\|y(t) - \theta\|}\left(2 - \frac{\lambda_{\max}}{\|y(t) - \theta\|}\right)(y(t) - \theta)^T \Sigma^{-1}(y(t) - \theta)
$$

$$
\leq -\frac{\lambda_{\max}}{\|y(t) - \theta\|}(y(t) - \theta)^T \Sigma^{-1}(y(t) - \theta)
$$

$$
\leq -\frac{\lambda_{\max}}{\|y(t) - \theta\|}\|y(t) - \theta\|^2 \lambda_{\max}^{-1}
$$

$$
\leq -\|y(t) - \theta\|
$$

$$
\leq -\min\{\sqrt{\lambda_{\min}/8}, 1/2\}\|\theta\|.
$$

Denote $H(t)$ to be the ball centered at $y'(t)$ with radius $(\lambda_{\min}/8)\min\{\sqrt{\lambda_{\min}/8}, 1/2\}$. Then for any $x \in H(t)$, we have

$$
(x - \theta)^T \Sigma^{-1}(x - \theta) - (y(t) - \theta)^T \Sigma^{-1}(y(t) - \theta)
$$

$$
= (y'(t) - \theta)^T \Sigma^{-1}(y'(t) - \theta) + 2(x - y'(t))^T \Sigma^{-1}(y'(t) - \theta) + (x - y'(t))^T \Sigma^{-1}(x - y'(t))
$$

$$
- (y(t) - \theta)^T \Sigma^{-1}(y(t) - \theta)
$$

$$
\leq (y'(t) - \theta)^T \Sigma^{-1}(y'(t) - \theta) + 2\lambda_{\min}^{-1}\|x - y'(t)\|\|y'(t) - \theta\| + \lambda_{\min}^{-1}\|x - y'(t)\|^2
$$

$$
- (y(t) - \theta)^T \Sigma^{-1}(y(t) - \theta)
$$

$$
\leq -\min\{\sqrt{\lambda_{\min}/8}, 1/2\}\|\theta\| + 2\lambda_{\min}^{-1}\|x - y'(t)\|\|y'(t) - \theta\| + \lambda_{\min}^{-1}\|x - y'(t)\|^2
$$

$$
\leq -\min\{\sqrt{\lambda_{\min}/8}, 1/2\}\|\theta\| + 4\lambda_{\min}^{-1}\|x - y'(t)\|\|\theta\| + \lambda_{\min}^{-1}\|x - y'(t)\|^2
$$

$$
\leq -\frac{1}{2}\min\{\sqrt{\lambda_{\min}/8}, 1/2\}\|\theta\| + \frac{\lambda_{\min}}{64}(\min\{\sqrt{\lambda_{\min}/8}, 1/2\})^2
$$

$$
\leq 0,
$$

where the last inequality holds under the assumption $\|\theta\| \geq \frac{\lambda_{\min}}{32}\min\{\sqrt{\lambda_{\min}/8}, 1/2\}$. Hence, $H(t) \subset \mathcal{S}_2(s(t); t)$ and consequently $H(t) \subset D(t)$. On the other hand, for any $x \in H(t)$, we have

$$
\|x\| \leq \|x - y'(t)\| + \|y'(t) - y(t)\| + \|y(t)\|
$$

$$
\leq (\lambda_{\min}/8)\min\{\sqrt{\lambda_{\min}/8}, 1/2\} + \lambda_{\max} + s(t).
$$

The proof is complete. $\qquad\square$

**Lemma C.10.** *Assume $d = O(1)$. Consider any $a \neq b \in [k]$. Assume there exist constants $\lambda_{\min}, \lambda_{\max}$ such that $0 < \lambda_{\min} \leq \lambda_1(\Sigma_j^*) \leq \lambda_d(\Sigma_j^*) \leq \lambda_{\max}$ for any $j \in \{a, b\}$. Then $SNR'_{a,b}$ and $\|\Xi_{b,a}\|$ are of the same order. When $SNR'_{a,b} = O(1)$, we have $P_{a,b}(0) \geq c$ for some constant $c > 0$. When $SNR'_{a,b} \to \infty$, we have*

$$
P_{a,b}(0) \geq \exp\left(-\frac{1 + o(1)}{8}SNR'^2_{a,b}\right),
$$

*and for any $\delta = o(1)$, we have*

$$
P_{a,b}(\delta) \leq \exp\left(-\frac{1 - o(1)}{8}SNR'^2_{a,b}\right).
$$

*Proof.* Recall the setting stated in Lemma C.9. By the definition of $B_{a,b}$, we can take $\theta = (\Sigma_a^*)^{-\frac{1}{2}}\Xi_{b,a}$, $\Sigma = (\Sigma_a^*)^{-\frac{1}{2}}\Sigma_b^*(\Sigma_a^*)^{-\frac{1}{2}}$, and $t = \log|(\Sigma_a^*)^{-\frac{1}{2}}\Sigma_b^*(\Sigma_a^*)^{-\frac{1}{2}}|$ such that $B_{a,b} = D(t)$. Due to $d = O(1)$ and (24), we have that all eigenvalues of $\Sigma$ are constants and that $\log|(\Sigma_a^*)^{-\frac{1}{2}}\Sigma_b^*(\Sigma_a^*)^{-\frac{1}{2}}|$ is also a constant. Consequently, $t$ is constant. In addition, $\|\theta\|$ is of the same order as $\|\Xi_{b,a}\|$.

When $\|\Xi_{b,a}\|$ is a constant, $\|\theta\|$ is a constant, then $\mathrm{SNR}'_{a,b}$ must be a constant as well. This is because $D(t)$ is the set of points where one density is greater or equal to another. Then $D(t)$ is non-empty as both densities have integral 1. Hence, $s(t)$ must be finite, meaning $\mathrm{SNR}'_{a,b}$ is a constant. In this case, we have $P_{a,b}(0) = \mathbb{P}(\eta \in D(t))$ being a constant as well.

When $\|\Xi_{b,a}\| \to \infty$, we have $\|\theta\| \to \infty$. Since all the assumptions needed in Lemma C.9 are satisfied, by its first conclusion, we have $s(t)$ being of the same order as $\|\theta\|$, and consequently being of the same order as $\|\Xi_{b,a}\|$. As a result, $\mathrm{SNR}'_{a,b}$ and $\|\Xi_{b,a}\|$ are of the same order. In addition, $H(t)$ exists and its radius is some constant $c_1 > 0$. Hence, its volume is $c_1^d V_d$ where $V_d$ is denoted as the volume of a $d$-dimensional unit ball. In addition, for any $x \in H(t)$, we have $\|x\| \le s(t) + c_2 = \mathrm{SNR}'_{a,b}/2 + c_2$ for some constant $c_2 > 0$. Recall $\eta \sim N(0, I_d)$. Then we have

$$
\begin{aligned}
P_{a,b}(0) \ge \mathbb{P}(\eta \in H(t)) &\ge c_1^d V_d \min_{x \in H(t)} \frac{1}{\sqrt{(2\pi)^d}} \exp\left(-\frac{1}{2}\|x\|^2\right) \\
&\ge c_1^d V_d \frac{1}{\sqrt{(2\pi)^d}} \exp\left(-\frac{1}{2}\left\|\mathrm{SNR}'_{a,b}/2 + c_2\right\|^2\right) \\
&\ge \exp\left(-\frac{1+o(1)}{8}\mathrm{SNR}'^{2}_{a,b}\right).
\end{aligned}
$$

Now let us consider $B_{a,b}(\delta)$ where $\delta = o(1)$. We can take $\theta, \Sigma$ same as before, but let $t' = \log |(\Sigma_a^*)^{-\frac{1}{2}} \Sigma_b^* (\Sigma_a^*)^{-\frac{1}{2}}| - \delta \Xi_{b,a}^T (\Sigma_b^*)^{-1} \Xi_{b,a}$. Then we have $t' = o(1)\|\theta\|^2$. Hence, by Lemma C.9, we have

$$
\left|\mathrm{SNR}'_{a,b} - \mathrm{SNR}'_{a,b}(\delta)\right| = 2\left|s(t) - s(t')\right| \preceq \frac{|t - t'|}{\|\theta\|} = o(1)\|\theta\| = o(1)\mathrm{SNR}'_{a,b}.
$$

Hence,

$$
\begin{aligned}
P_{a,b}(\delta) = \mathbb{P}(\eta \in B_{a,b}(\delta)) &\le \max_{x \in B_{a,b}(\delta)} \frac{1}{\sqrt{(2\pi)^d}} \exp\left(-\frac{1}{2}\|x\|^2\right) \\
&= \frac{1}{\sqrt{(2\pi)^d}} \exp\left(-\frac{1}{2}\min_{x \in B_{a,b}(\delta)}\|x\|^2\right) \\
&= \frac{1}{\sqrt{(2\pi)^d}} \exp\left(-\frac{1}{8}\mathrm{SNR}'^{2}_{a,b}(\delta)\right) \\
&= \exp\left(-\frac{1-o(1)}{8}\mathrm{SNR}'^{2}_{a,b}\right).
\end{aligned}
$$

$\square$

